# BiScope: AI-generated Text Detection by Checking Memorization of Preceding Tokens

**Hanxi Guo**
Purdue University
guo778@purdue.edu

**Siyuan Cheng**
Purdue University
cheng535@purdue.edu

**Xiaolong Jin**
Purdue University
jin509@purdue.edu

**Zhuo Zhang**
Purdue University
zhan3299@purdue.edu

**Kaiyuan Zhang**
Purdue University
zhan4057@purdue.edu

**Guanhong Tao**
University of Utah
guanhong.tao@utah.edu

**Guangyu Shen**
Purdue University
shen447@purdue.edu

**Xiangyu Zhang**
Purdue University
xyzhang@cs.purdue.edu

## Abstract

Detecting text generated by Large Language Models (LLMs) is a pressing need in order to identify and prevent misuse of these powerful models in a wide range of applications, which have highly undesirable consequences such as misinformation and academic dishonesty. Given a piece of subject text, many existing detection methods work by measuring the difficulty of LLM predicting the next token in the text from their prefix. In this paper, we make a critical observation that how well the current token's output logits memorizes the closely preceding input tokens also provides strong evidence. Therefore, we propose a novel bi-directional calculation method that measures the cross-entropy losses between an output logits and the ground-truth token (forward) and between the output logits and the immediately preceding input token (backward). A classifier is trained to make the final prediction based on the statistics of these losses. We evaluate our system, named BiScope, on texts generated by five latest commercial LLMs across five heterogeneous datasets, including both natural language and code. BiScope demonstrates superior detection accuracy and robustness compared to nine existing baseline methods, exceeding the state-of-the-art non-commercial methods' detection accuracy by over $0.30$ F1 score, achieving over $0.95$ detection F1 score on average. It also outperforms the best commercial tool GPTZero that is based on a commercial LLM trained with an enormous volume of data. Code is available at https://github.com/MarkGHX/BiScope.

## 1 Introduction

Given the superior performance of Large Language Models (LLMs) in understanding and generating text, they have become an integral part of human society, assisting people with daily activities such as summarizing articles, polishing emails, and more. However, the widespread use of LLMs also raises concerns about the misuse of AI-generated text. For instance, students and academics may utilize LLMs to produce content for their assignments and research [10, 37, 30], compromising academic integrity. Adversarial individuals could leverage LLMs to efficiently create inflammatory and fraudulent content on social media [22]. Additionally, the development of LLMs themselves faces challenges related to the quality of existing datasets, which may be compromised by the significant

inclusion of AI-generated text [49, 27]. All of these issues underscore the urgent need to distinguish AI-generated text from human-written text [5, 14, 18, 50, 3, 12].

Despite this urgency, current AI-generated text detection techniques fall short as LLMs become increasingly diverse and advanced. Our experiments demonstrate that most of the existing approaches cannot achieve an F1 score exceeding 80% on the Yelp dataset [32] when using the latest LLMs (e.g., Claude-3-Opus) to generate content. A close examination of these approaches reveals inherent limitations by design. Specifically, there are three kinds of methods that do not need pre-training or additional information on the LLMs that generate the data (*e.g.*, watermarking [7, 19, 44, 46, 20]). The first kind [5] directly prompts another LLM or NLP model to classify whether the subject text is AI-generated. While intuitive, the inevitable model hallucination [55, 26] consistently prevents it from achieving a high accuracy. The second kind of methods [28, 31, 9, 54] examines the linguistic features of the subject text, which are increasingly susceptible to deception as LLMs become more sophisticated and human-like in their responses. The third approach [11, 41, 17, 40, 16, 32, 35, 4, 53, 48] feeds partial or entire text to a surrogate model and checks how well the output text aligns with the surrogate model's preference via various metrics or downstream classifiers. While this method outperforms the first two approaches, it only examines the next token information in the output logits, representing just part of model behaviors, thereby naturally limiting its performance.

In this work, we explore the potential of leveraging internal model states to detect AI-generated text. Like existing methods, we hypothesize that since LLMs are trained on vast corpora of data from the Internet, their training data likely exhibit significant similarities, leading to similar behaviors across models. Therefore, we use a surrogate model to approximate the behaviors of the one used to generate the subject text. We also make a critical observation that, in causal language models (*e.g.*, GPT), the current token's output logits encode information about both the next token (*i.e.*, prediction) and its preceding input tokens (*i.e.*, memorization), indicating a bidirectional relationship between the output logits and the input text. Specifically, when these causal language models encounter human data, they tend to memorize more preceding token information while predicting less next token information in their output logits.

To reveal this relationship, we calculate two kinds of cross-entropy losses by feeding different portions of the subject text into the surrogate model. One is the forward information, calculated as the cross-entropy loss between the output logits and the expected next token in the subject text. The other is the backward information, calculated as the cross-entropy loss between the output logits and the most preceding input token. We then train a binary classifier on the collected statistical loss features to make the final prediction. We also introduce several novel improvements in the prototype, such as providing a summary of the subject text to better guide the surrogate model, thereby enhancing its practical effectiveness and robustness, and using parallel model inference to enhance efficiency. As such, we propose BiScope, an effective and efficient AI-generated text detector by harnessing both the prediction and memorization features of the LLM through its output logits.

Our contributions are as follows:

- We propose a novel AI-generated text detection algorithm that exploits both the preceding token information (*i.e.*, memorization) and the next token information (*i.e.*, prediction) via an innovative bi-directional cross-entropy loss calculation method. Additionally, we are the first to utilize text summaries to guide the detection, further enhancing its effectiveness and robustness toward heterogeneous data.

- We extend existing datasets and craft a large-scale public dataset for more challenging AI-generated texts, consisting of 25 distinct groups and more than $22,000$ samples. The dataset is sourced from five different text domains (both natural language and code) and generated by the five latest commercial LLMs. This dataset presents more challenging scenarios compared to existing datasets, which are typically sourced from open-source LLMs with fewer parameters and capabilities. We also craft a paraphrased version of our dataset.

- We develop a prototype named BiScope, a detection pipeline without any fine-tuning needed for the detection LLM. We evaluate it on our dataset and compare it with nine state-of-the-art baseline techniques. Our results show that BiScope can achieve an average F1 score of over $0.95$, taking less than 200 milliseconds to detect a sample (when the summary procedure is disabled), while the baseline techniques achieve only $0.70$-$0.85$ F1 scores and take up to 27 seconds per sample. BiScope also outperforms the best commercial tool,

GPTZero, in 72% of cases. Additionally, we conduct a comprehensive ablation study to verify the effectiveness and robustness of each component of BiScope.

## 2  Background and Related Work

In addition to various watermarking techniques [7, 19, 44, 46, 20, 24, 15, 52] that require fine-tuning or additional information about the LLMs generating the text, several efforts have been directed towards the detection of AI-generated texts with minimal prior knowledge about the generative models. These efforts broadly fall into three categories. As stated in § 1, the first two categories perform worse than the last category, hence we mainly focus on the methods in the third category that use a surrogate LLM in this paper. The methods in the third category can be further divided into two types: statistical methods and training-based methods.

**Statistical Methods.** These techniques [45, 34, 21, 33] primarily utilize pre-trained LLMs to simulate the generation process of the target generative AI, analyzing the statistical differences between AI-generated texts and human-written ones. These methods commonly serve as zero-shot approaches, assigning scores to indicate the probability of texts being AI-generated. For example, Zero-shot Query [32, 48] prompts a pre-trained LLM to score the input text. LogRank [11, 35] calculates the average probability rank of each token in a text processed through a pre-trained LLM, where higher ranks suggest the text is AI-generated. LRR [41] improves on LogRank by incorporating token confidence. DetectGPT [35] involves masking parts of the text to see how an LLM reconstructs them, and Raidar [32] employs the LLM to rewrite the text. Both methods assume that AI-generated texts are more likely to be preserved accurately in the process. Binoculars [13] analyzes the cross-entropy between the output logits from two surrogate models with different fine-tuning configurations.

**Training-based Methods.** This type includes methods train an NLP model to distinguish between AI-generated and human-written texts. For example, OpenAI [40] uses a RoBERTa-based model for training an AI-text classifier. Such methods can be susceptible to adversarial attacks or paraphrasing. The state-of-the-art technique RADAR [16] leverages adversarial training to improve the robustness of the classifier. There are also commercial services for the detection of AI-generated texts. For example, **GPTZero** [43] employs a multi-step statistical detection process and utilizes a pre-trained commercial LLM to deliver the prediction.

Our approach, BiScope, is a statistical method that uniquely incorporates meticulous statistical feature extraction via a bi-directional calculation method to ensure its general effectiveness and robustness against paraphrasing across five text domains and five of the latest commercial LLMs. To evaluate BiScope, we compare it with nine existing detection methods, including Zero-shot Query [48, 32], LogRank [11], LRR [41], DetectGPT [35], RADAR [16], Raidar [32], OpenAI Detector [40], Binoculars [13], and GhostBuster [48], as well as with the most renowned commercial detection API, **GPTZero**. We surpass these baseline methods in both effectiveness and efficiency.

## 3  Methodology of BiScope

### 3.1  Design Motivation

Although existing AI-generated text detection methods have been proven to be robust against texts generated by various open-source LLMs, their performance degrades in more complex and real-world scenarios, especially when dealing with the latest commercial LLMs and heterogeneous text genres. The degradation can be attributed to the following two reasons: **feature insufficiency** and **contextual heterogeneity**.

**Feature Insufficiency.** Existing methods focus on analyzing the difficulty a surrogate LLM experiences in predicting the next token given the preceding input text. For example, these methods use the rank or the probability of the next token as a metric or compare the discrepancy between the input text and the surrogate model's generation. Figure 1(a) presents the detection F1 score of a toy example that uses the average next token rank from Llama-2-7B as the feature and a random forest model as the classification model on both human text (in blue) and GPT-4-Turbo's text (in orange). The detection F1 score only reaches 0.55, which is just slightly better than random guesses. This indicates the lack of a clear separation using only next token ranks. One may argue that the random forest may

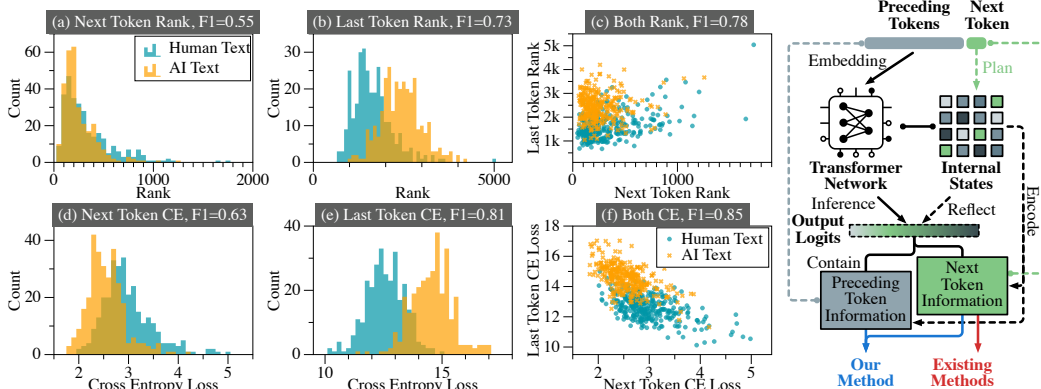

**Figure 1:** Comparison of detection F1 scores when utilizing the rank and cross-entropy loss regarding next token, preceding token or both. The surrogate detection model is Llama-2-7B.

**Figure 2:** Comparison of output logits information utilization.

not be powerful enough. However, we will show later that using additional features proposed in the paper, the same random forest configuration could achieve much better results.

We observe that the internals of the surrogate LLM when used to predict the subject text have much richer information that can be used. Figure 2 illustrates how LLM encodes information. The arrows and texts in green illustrate the information related to the next token, while the arrows and texts in gray show the information related to the preceding tokens. In the auto-regressive generation mode, the LLM receives the tokens preceding to the current position as the input and outputs the logits that contains its prediction for the next token. During this procedure, its internal states encode the preceding tokens (i.e., memorization) [47] while implicitly "planning" for the next token [51], namely, as observed by researchers in [51], the internal states show similarities to the encodings of future tokens. The output logits, which can be considered as a reduced representation of the model's internal states, also contains both the information to predict the next token and the information of preceding tokens. Existing methods focus only on the former by comparing the output logits with the expected next token. In this paper, we propose to consider the preceding token information as well. In particular, we hypothesize the following: **for human-written text, the surrogate LLM has a poor prediction for the next token and a strong memory of the previous token, reflected in the output logits, whereas the behaviors for LLM-generated text are the opposite.** Intuitively, it's like when we humans are unsure of what to say next, and the last word tends to stay in our minds.

To validate our hypothesis, we conducted an experiment in which we compared the current output logits with the preceding token for a piece of a given text, leveraging the same random forest as before. Figure 1(b) and (c) present the detection F1 score when only using the preceding token's rank and when using both the preceding and next tokens' ranks (to distinguish human and LLM texts), reaching 0.73 and 0.78 F1 scores, respectively, denoting a 0.2 F1 score improvement compared to using the next token information alone. Observe that in (d) for next token prediction, the AI texts (in orange) tend to have a smaller CE loss than human texts (in blue), indicating the LLM has a better prediction for the AI texts. In (e) for previous token memorization, the human texts tend to have a smaller loss than the AI texts, indicating the LLM has poorer memory for AI texts. Figures (a) and (b) and an additional example in Appendix B show a similar trend. These support our hypothesis.

Thus, in BISCOPE, we design a novel bi-directional cross-entropy loss computation method that computes the cross-entropy losses between the output logits and the expected next token, and between the output logits and the preceding token. Figure 1(d)-(f) illustrate the F1 scores when using the cross-entropy losses for next token, previous token, and both. Observe they achieve better results compared to using plain ranks, due to the more wealthy information encoded. An additional example using GPT-Neo-2.7B in Appendix B shows a similar trend.

**Contextual Heterogeneity.** In addition to insufficient feature utilization, we also observe that contextual heterogeneity significantly influences detection accuracy. Existing methods directly use surrogate LLMs to generate the given text in an auto-regressive manner, without incorporating any

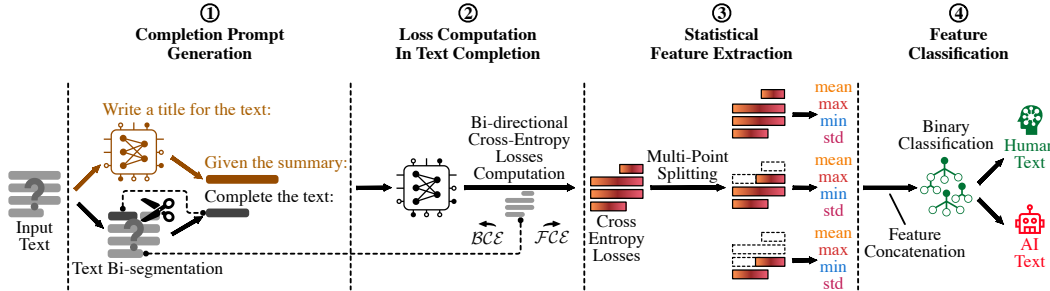

**Figure 3: Overview of BISCOPE. Arrows and texts in brown indicate text summarization.**

additional information of the context (for the text). As such, given a prefix part of the text, the LLM may have a diverse set of possible completions, limiting the ability to separate human and LLM texts.

To alleviate this problem, we formalize our detection as a guided completion task, using a surrogate LLM to first summarize the entire input text. These text summaries are then used to guide the completion, providing complementary contextual information and making the features more robust.

### 3.2 Overview of BISCOPE

The entire workflow of BISCOPE can be summarized in four key steps, shown in Figure 3.

Step ①: **Completion Prompt Generation.** In the first step, we initialize the detection as a guided text completion task. We use a surrogate LLM to summarize the input text and generate a text summary as a guidance. We then divide the input text into two segments. The first segment, along with a completion request, is utilized to construct a text completion request. The text summary guidance and the text completion request form a completion prompt. Details are presented in § 3.3.

Step ②: **Loss Computation In Text Completion.** Given the completion prompt and the second segment of the input text from Step ①, we then calculate our novel bi-directional cross-entropy losses for the tokens in the second text segment using multiple open-source LLMs in parallel. Details are presented in § 3.4. The use of multiple LLMs is to reduce the uncertainty.

Step ③: **Statistical Feature Extraction.** We vary the separation of the two segments at different positions of the subject text (e.g., one-fourth, half, and three-fourth of the whole length). For each setup, we collect the statistics of the bi-directional cross-entropy loss values. The statistics are concatenated to form a feature vector. More details and justifications are presented in § 3.5.

Step ④: **Feature Classification.** In the final step, we use the concatenated feature vector to train a binary classifier, which determines whether the input text is human-generated or AI-generated. Further details are presented in § 3.6.

### 3.3 Completion Prompt Generation

In BISCOPE, we calculate the bi-directional cross-entropy losses within a guided text completion scenario. This scenario involves providing a text summary guidance and a short sub-string of the input text to force LLMs to generate the remainder of the text. Specifically, to alleviate the impact of contextual heterogeneity during text generation, we first utilize a surrogate LLM to summarize the entire input text and obtain a summary as guidance. We then divide the input text into two segments (e.g., the first 10% and the remaining 90%). The first segment, referred to as `Input Text Segment 1`, serves as the sub-string in a text completion request, while the second segment, referred to as `Input Text Segment 2`, is used as the completion ground-truth in § 3.4. By appending the text completion request after the summary guidance, we construct a completion prompt shown as follows:

```
Given the summary:
{Text Summary Guidance}
Complete the following text:
{Input Text Segment 1}
```

The text in black indicates the text completion request, while the text in brown indicates the guidance, which is summarized using the following prompt:

```
Write a title for this text: {Input Text}
```

The summary contains the aggregated contextual information of the entire text, providing complementary guidance to the LLM completion, in addition to the first segment. To balance the detection accuracy and efficiency, BISCOPE can also disable this text summary procedure to achieve faster AI-generated text detection with satisfactory accuracy.

### 3.4 Loss Computation In Text Completion

After crafting the completion prompt, we then feed it into multiple open-source LLMs in parallel to obtain multiple output logits that correspond to the `Input Text Segment 2` from § 3.3 using the *teacher forcing pattern* [25], which feeds the ground-truth token prefixes (`Input Text Segment 2`) to compute the output logits at each token position. These output logits can be used to measure how likely the LLMs predict the next token given its prefix in the subject text and how well the LLMs memorize the preceding token, according to our discussion in § 3.1. We hence propose a bi-directional cross-entropy calculation method in BISCOPE, which consists of both forward and backward cross-entropy calculations. The forward cross-entropy ($\mathcal{FCE}$) calculation is identical to the commonly used cross-entropy in most LLM training processes, utilizing the output logits and the next ground-truth token to capture the output logits' next-token-related information. In contrast, the backward cross-entropy ($\mathcal{BCE}$) is calculated between the output logits and the immediate preceding input token, capturing how much the logits memorizes the preceding token. The detailed $\mathcal{FCE}$ and $\mathcal{BCE}$ calculations at token position $i$ with LLM $\mathcal{M}$ are shown in Equation 1:

$$\mathcal{FCE}_i = -\sum_{z=1}^{||\mathcal{V}||} \tilde{\mathcal{P}}_{i+1}^z \cdot \log(\mathcal{P}_i^z), \quad \mathcal{BCE}_i = -\sum_{z=1}^{||\mathcal{V}||} \tilde{\mathcal{P}}_i^z \cdot \log(\mathcal{P}_i^z) \tag{1}$$

where $\mathcal{V}$ indicates the vocabulary of the LLM, $\mathcal{P}_i$ denotes the soft-maxed output logits from $\mathcal{M}$ at position $i$ of the generated text (when given the preceding tokens from the completion prompt). $\tilde{\mathcal{P}}_i$ indicates the ground-truth token encoding at the same position.

### 3.5 Statistical Feature Extraction

The aforementioned bi-directional cross-entropy loss values may have different characteristics when the text is partitioned at different positions. Intuitively, when the text is partitioned at a ratio of 1:9, meaning that we use the first 10% of the text to perform the completion, there tends to be a lot more uncertainty compared to a partition of 9:1. A naive design is to fix a partition ratio. However, finding the most effective partition is difficult. Another design is to use the loss values computed at all positions. However, it can hardly deal with length variations of input texts. Therefore, our design is to partition the whole text into $n$ segments ($n = 10$ in our implementation). For each segment, we collect the bidirectional loss value statistics over all the positions with the segment, including the *mean*, *maximum*, *minimum*, and *standard deviation* values. This allows us to align texts of various lengths and leverage the later classification to figure out the best partition positions (through learning). Additionally, this multi-segment analysis requires only a one-time inference of the input text, as the loss calculation at each position is independent of the others via teacher forcing. This allows BISCOPE to obtain various and sufficient features with high efficiency.

### 3.6 Feature Classification

In the final step, we concatenate all the statistical features of both the $\mathcal{FCE}$ and $\mathcal{BCE}$ vectors from all the detection LLMs into a one-dimensional feature vector, which is then used to train a binary classifier to perform the classification. Due to the generality of these features, the binary classifier can be directly used to detect unseen data, whether from unknown LLMs or unfamiliar text domains.

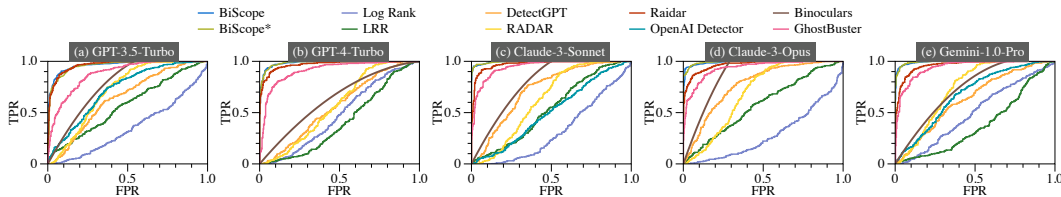

Figure 4: ROC curves of BISCOPE and all the baselines.

## 4 Evaluation Results

### 4.1 Experimental Setup

We use five datasets in our evaluation, including two short natural language datasets (Arxiv [32] and Yelp [32]), two long natural language datasets (Creative [48] and Essay [48]), and one code dataset [8]. For all datasets, we reuse their human-generated data and craft AI-generated text using five of the latest commercial LLMs. More details are presented in Appendix C. To the best of our knowledge, we are the first to craft a comprehensive AI-generated text dataset using five of the latest commercial LLMs from three leading AI corporations. For the metrics, we use two metrics in our evaluation. To assess the effectiveness of the detection, we use a 5-fold cross-validation F1 score. To evaluate the efficiency of the detection, we use the time cost per sample as the metric. We present more hyper-parameter settings in Appendix A.

### 4.2 Detection Performance Comparison with Existing Baselines

We first evaluate BISCOPE (with and without the text summary guidance) and nine detection methods on the normal dataset under both in-distribution and out-of-distribution (OOD) settings. Under the in-distribution setting, the test data and training data are from the same source. While under the OOD setting, the test data is from an unknown source. The results are presented in Table 1. Our BISCOPE outperforms existing AI-generated text detection methods, achieving a 0.25 average detection F1 score increase on the normal dataset under in-distribution setting. In the OOD setting, our BISCOPE still surpasses existing detection methods with over a 0.16 average detection F1 score increase. Detailed analysis is shown as follows.

**In-Distribution Results.** For the in-distribution setting, we report a 5-fold cross-validation F1 score using one piece of human data and one piece of AI-generated data from the five latest commercial LLMs, as shown in Table 1. BISCOPE * indicates our method with text summary guidance, while BISCOPE represents our method without it. Our method outperforms all nine baselines across all five datasets generated by the latest LLMs. Specifically, our method achieves a 0.29 average F1 score increase on the two short natural language datasets, where existing methods' best F1 score is around 0.90, while BISCOPE reaches over 0.95 in most cases. On the two long natural language datasets, existing methods reach up to 0.97, while BISCOPE achieves over 0.99 in all cases, resulting in a 0.23 average improvement. On the code dataset, existing methods achieve a 0.60-0.70 average, whereas BISCOPE reaches 0.84, achieving a 0.21 average increase. We further present the TPR-FPR (ROC) curves of BISCOPE and 8 baselines in Figure 4 on the Yelp dataset. We observe that our BISCOPE reaches over 0.8 detection TPR on average when the FPR is only 0.01, outperforming all the baselines on all the five generative models' data.

**Out-of-Distribution Results.** We use two out-of-distribution (OOD) evaluation settings on the normal dataset, cross-model (CM) and cross-dataset (CD), based on previous studies [48, 32]. In CM setting, we assess detection performance on AI-generated data from unknown LLMs within the same text domain by training classifiers on human data and one piece of AI-generated data, then testing on four others from the same dataset. In CD setting, we evaluate detection transferability across text domains by training on data from one dataset and testing on data from four others, all generated by the same LLM. Results across five LLMs are reported in Table 1. In the CM setting, the highest baseline F1 score is 0.84 (average 0.77), while our BISCOPE achieves 0.93, showing a 0.16 average increase. This indicates superior generality of BISCOPE in detecting unseen LLM-generated texts. In the CD setting, BISCOPE's average F1 score is 0.05 lower than RADAR that utilizes a model

Table 1: Detection performance of BISCOPE and nine baselines on both normal and paraphrased datasets with in-distribution and out-of-distribution (OOD) settings.

| | Method | Normal Dataset | | | | | Paraphrased Dataset | | | | Normal OOD Avg.-CM | Normal OOD Avg.-CD | Paraphrased OOD Avg. |
|---|---|---|---|---|---|---|---|---|---|---|---|---|---|
| | | GPT-3.5 Turbo | GPT-4 Turbo | Claude-3 Sonnet | Claude-3 Opus | Gemini 1.0-pro | GPT-3.5 Turbo | GPT-4 Turbo | Claude-3 Sonnet | Claude-3 Opus | | | |
| Arxiv | Zero-shot Query | 0.5768 | 0.5835 | 0.6764 | 0.6667 | 0.6666 | 0.5587 | 0.6116 | 0.6916 | 0.6935 | - | - | - |
| | Log Rank | 0.6572 | 0.7006 | 0.8015 | 0.8809 | 0.8560 | 0.6628 | 0.6660 | 0.6634 | 0.6747 | 0.6913 | 0.6219 | 0.3655 |
| | LRR | 0.6602 | 0.7031 | 0.8116 | 0.8596 | 0.8544 | 0.6654 | 0.6654 | 0.6654 | 0.6654 | 0.7319 | 0.6130 | 0.2353 |
| | DetectGPT | 0.6654 | 0.6634 | 0.6673 | 0.6673 | 0.6673 | 0.6641 | 0.6628 | 0.6654 | 0.6654 | 0.6642 | 0.7091 | 0.6352 |
| | RADAR | 0.9566 | 0.7858 | 0.7034 | 0.7754 | 0.7868 | 0.9203 | 0.6970 | 0.6884 | 0.7202 | 0.7404 | 0.8035 | 0.7388 |
| | Raidar | 0.8316 | 0.8157 | 0.8029 | 0.8289 | 0.7366 | 0.9004 | 0.8851 | 0.8052 | 0.8303 | 0.6984 | 0.6524 | 0.7270 |
| | OpenAI Detector | 0.7889 | 0.6660 | 0.6673 | 0.6673 | 0.6976 | 0.7062 | 0.6654 | 0.6673 | 0.6673 | 0.6249 | 0.6569 | 0.6705 |
| | Binoculars | 0.9097 | 0.9135 | 0.9256 | 0.9699 | 0.9560 | 0.6617 | 0.6971 | 0.8112 | 0.8672 | 0.9163 | **0.8199** | 0.5835 |
| | GhostBuster | 0.9716 | 0.9886 | 0.9815 | 0.9813 | 0.9571 | 0.9700 | **0.9943** | **0.9814** | 0.9856 | 0.9187 | 0.6811 | **0.9672** |
| | BISCOPE | 0.9870 | 0.9928 | 0.9796 | 0.9885 | 0.9708 | 0.9769 | 0.9800 | 0.9625 | 0.9870 | 0.9517 | 0.7131 | 0.8534 |
| | BISCOPE* | **0.9928** | **0.9943** | **0.9869** | **0.9913** | **0.9797** | **0.9870** | 0.9859 | 0.9593 | **0.9884** | **0.9775** | 0.7767 | 0.8740 |
| Yelp | Zero-shot Query | 0.0020 | 0.0010 | 0.0110 | 0.0168 | 0.0080 | 0.0000 | 0.0009 | 0.0188 | 0.0188 | - | - | - |
| | Log Rank | 0.6776 | 0.6721 | 0.7120 | 0.6946 | 0.6439 | 0.6754 | 0.6660 | 0.6745 | 0.6743 | 0.6574 | 0.6695 | 0.6258 |
| | LRR | 0.6671 | 0.6678 | 0.6733 | 0.6678 | 0.6358 | 0.6681 | 0.6662 | 0.6674 | 0.6666 | 0.6589 | 0.6615 | 0.6508 |
| | DetectGPT | 0.6945 | 0.6737 | 0.7252 | 0.7477 | 0.6626 | 0.6702 | 0.6669 | 0.7009 | 0.7166 | 0.6738 | 0.7187 | 0.6710 |
| | RADAR | 0.7618 | 0.7090 | 0.7310 | 0.7590 | 0.7497 | 0.7485 | 0.7030 | 0.7117 | 0.7345 | 0.7148 | 0.7370 | 0.7033 |
| | Raidar | **0.9023** | 0.8985 | 0.9180 | 0.8876 | 0.8915 | 0.8948 | 0.9124 | 0.9344 | 0.9128 | 0.8572 | 0.6850 | 0.7817 |
| | OpenAI Detector | 0.7286 | 0.6668 | 0.6668 | 0.6616 | 0.6798 | 0.7240 | 0.6668 | 0.6668 | 0.6668 | 0.6348 | 0.6308 | 0.6563 |
| | Binoculars | 0.7295 | 0.6665 | 0.7583 | 0.8260 | 0.6885 | 0.6683 | 0.6655 | 0.6908 | 0.7284 | 0.6930 | 0.8474 | 0.6681 |
| | GhostBuster | 0.8193 | 0.8369 | 0.8746 | 0.8644 | 0.8625 | 0.8174 | 0.8649 | 0.9271 | 0.9145 | 0.7975 | 0.5859 | 0.8452 |
| | BISCOPE | **0.9023** | 0.9405 | 0.9652 | 0.9532 | 0.9486 | 0.9064 | 0.9473 | 0.9814 | **0.9789** | 0.9063 | **0.8608** | **0.9523** |
| | BISCOPE* | 0.9010 | **0.9452** | **0.9658** | **0.9570** | **0.9545** | **0.9102** | **0.9530** | **0.9830** | 0.9757 | **0.9128** | 0.8455 | 0.9505 |
| Creative | Zero-shot Query | 0.2730 | 0.1502 | 0.2691 | 0.3186 | 0.2719 | 0.2398 | 0.1694 | 0.1897 | 0.2948 | - | - | - |
| | Log Rank | 0.9673 | 0.7341 | 0.8779 | 0.9269 | 0.8044 | 0.7673 | 0.6685 | 0.6823 | 0.7701 | 0.7993 | 0.5824 | 0.5213 |
| | LRR | 0.9512 | 0.6732 | 0.8062 | 0.8884 | 0.7209 | 0.6638 | 0.6662 | 0.6649 | 0.6662 | 0.7242 | 0.5772 | 0.3846 |
| | DetectGPT | 0.8305 | 0.7090 | 0.7922 | 0.8166 | 0.7580 | 0.6850 | 0.6715 | 0.7364 | 0.7066 | 0.7573 | 0.5415 | 0.6209 |
| | RADAR | 0.9543 | 0.8869 | 0.9131 | 0.9345 | 0.9382 | 0.9298 | 0.8744 | 0.9160 | 0.9145 | 0.8934 | 0.7699 | 0.8937 |
| | Raidar | 0.8933 | 0.8303 | 0.8481 | 0.8661 | 0.8588 | 0.8271 | 0.8217 | 0.8120 | 0.7978 | 0.8151 | 0.7580 | 0.5621 |
| | OpenAI Detector | 0.6666 | 0.6671 | 0.6669 | 0.6671 | 0.6271 | 0.6671 | 0.6671 | 0.6671 | 0.6671 | 0.6103 | 0.6708 | 0.5475 |
| | Binoculars | 0.9945 | 0.9681 | 0.9814 | 0.9866 | 0.9880 | 0.9627 | 0.8381 | 0.9348 | 0.9540 | 0.9711 | 0.7599 | 0.8738 |
| | GhostBuster | 0.9965 | 0.9821 | 0.9834 | 0.9834 | 0.9920 | 0.9861 | 0.9786 | 0.9871 | 0.9865 | 0.9501 | **0.8206** | 0.9012 |
| | BISCOPE | **0.9985** | 0.9950 | **0.9960** | 0.9930 | 0.9964 | 0.9955 | 0.9945 | **0.9955** | 0.9940 | **0.9846** | 0.7980 | **0.9707** |
| | BISCOPE* | 0.9975 | **0.9955** | 0.9955 | **0.9945** | **0.9970** | 0.9955 | **0.9955** | 0.9950 | **0.9945** | 0.9780 | 0.8154 | 0.9513 |
| Essay | Zero-shot Query | 0.0156 | 0.0098 | 0.0175 | 0.0059 | 0.0322 | 0.0078 | 0.0214 | 0.0423 | 0.0078 | - | - | - |
| | Log Rank | 0.9936 | 0.9065 | 0.9771 | 0.9811 | 0.9774 | 0.9004 | 0.7067 | 0.8170 | 0.9313 | 0.9025 | 0.4518 | 0.6074 |
| | LRR | 0.9945 | 0.8416 | 0.9673 | 0.9836 | 0.9685 | 0.8121 | 0.6658 | 0.7243 | 0.8546 | 0.8911 | 0.4724 | 0.5395 |
| | DetectGPT | 0.9344 | 0.8709 | 0.9302 | 0.9378 | 0.9311 | 0.7910 | 0.7622 | 0.8609 | 0.8429 | 0.9140 | 0.5769 | 0.7513 |
| | RADAR | 0.9812 | 0.8978 | 0.9648 | 0.9555 | 0.9650 | 0.9509 | 0.8211 | 0.9471 | 0.9118 | 0.9386 | 0.7665 | 0.8883 |
| | Raidar | 0.9786 | 0.9424 | 0.9672 | 0.9710 | 0.9574 | 0.9448 | 0.9186 | 0.9146 | 0.9216 | 0.9416 | 0.7136 | 0.8000 |
| | OpenAI Detector | 0.7069 | 0.6664 | 0.6667 | 0.6669 | 0.6426 | 0.6669 | 0.6662 | 0.6667 | 0.6664 | 0.5870 | 0.6411 | 0.5300 |
| | Binoculars | 0.9995 | 0.9970 | 0.9945 | 0.9960 | 0.9978 | 0.9920 | 0.9607 | 0.9787 | 0.9955 | 0.9967 | 0.7383 | 0.9429 |
| | GhostBuster | 0.9995 | 0.9950 | 0.9960 | 0.9965 | 0.9967 | 0.9916 | 0.9861 | 0.9880 | 0.9930 | 0.9804 | **0.7740** | **0.9435** |
| | BISCOPE | **1.0000** | **0.9990** | 0.9985 | 0.9970 | **0.9994** | 0.9965 | **0.9990** | 0.9990 | 0.9980 | **0.9946** | 0.5456 | **0.9435** |
| | BISCOPE* | **1.0000** | **0.9990** | 0.9985 | 0.9975 | 0.9989 | 0.9975 | **0.9990** | 0.9985 | **0.9985** | 0.9914 | 0.5669 | 0.9292 |
| Code | Zero-shot Query | 0.6300 | 0.5833 | 0.4351 | 0.3524 | 0.1854 | 0.6690 | 0.6784 | 0.6400 | 0.4545 | - | - | - |
| | Log Rank | 0.6581 | 0.6610 | 0.6611 | 0.6569 | 0.6583 | 0.6612 | 0.6611 | 0.6556 | 0.6581 | 0.6521 | 0.5306 | 0.5539 |
| | LRR | 0.6639 | 0.6639 | 0.6639 | 0.6639 | 0.6542 | 0.6639 | 0.6639 | 0.6639 | 0.6639 | 0.6613 | 0.6475 | 0.6591 |
| | DetectGPT | 0.6361 | 0.6474 | 0.6583 | 0.6612 | 0.6682 | 0.6612 | 0.6639 | 0.6639 | 0.6612 | 0.6445 | 0.5936 | 0.6282 |
| | RADAR | 0.6680 | 0.6653 | 0.6652 | 0.6597 | 0.6626 | 0.6598 | 0.6653 | 0.7322 | 0.6653 | 0.6652 | **0.8114** | 0.6660 |
| | Raidar | 0.9368 | 0.8220 | 0.6121 | 0.6156 | 0.4858 | 0.9325 | 0.8744 | 0.8250 | 0.6197 | **0.8878** | 0.1378 | 0.6521 |
| | OpenAI Detector | 0.7213 | 0.6977 | 0.6916 | 0.6542 | 0.6666 | 0.7514 | 0.6639 | 0.6639 | 0.6695 | 0.6567 | 0.4083 | 0.5767 |
| | Binoculars | 0.7073 | 0.6512 | 0.6612 | 0.6653 | 0.6624 | 0.7101 | 0.6338 | 0.8041 | 0.7179 | 0.6273 | 0.7181 | 0.6771 |
| | GhostBuster | 0.8524 | 0.7942 | 0.6556 | **0.6749** | 0.3860 | 0.8662 | 0.7729 | 0.7757 | 0.5390 | 0.6232 | 0.5091 | 0.6790 |
| | BISCOPE | 0.9665 | **0.9655** | **0.8528** | 0.6069 | **0.7809** | **0.9659** | **0.9464** | **0.9691** | **0.9250** | 0.7974 | 0.5895 | 0.8999 |
| | BISCOPE* | **0.9692** | 0.9586 | 0.8526 | 0.6620 | 0.7741 | 0.9597 | 0.9435 | 0.9600 | 0.9222 | 0.7898 | 0.5855 | **0.9024** |

pre-trained on multiple datasets. However, compared to the other four baselines, BISCOPE still shows a 0.12 average improvement. Detailed OOD results are in Appendix D.

## 4.3 Robustness against Intentional Paraphrasing

Existing studies [23, 39] show that intentional paraphrasing can effectively evade AI-generated text detection. To verify the robustness of BISCOPE against paraphrasing, we utilize five of the latest commercial LLMs to paraphrase their own data using the paraphrasing prompt from previous work [16], and compare BISCOPE with nine baselines on this paraphrased dataset under both in-distribution and out-of-distribution settings, as shown in Table 1. Under the in-distribution setting, we test all the methods using the same configuration as on the normal dataset, exploring whether paraphrasing reduces the discriminative power of existing methods. BISCOPE outperforms existing

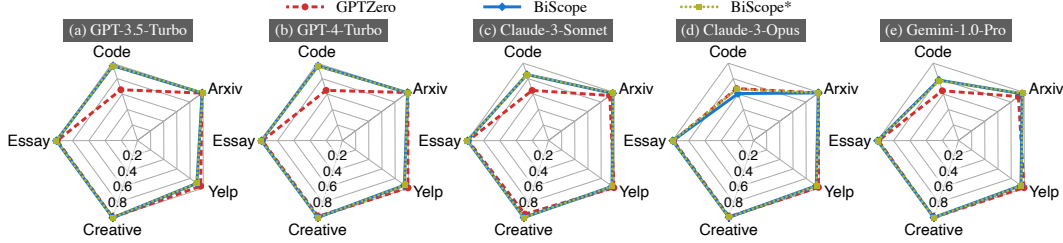

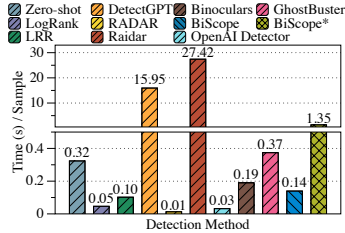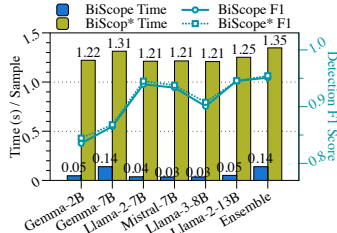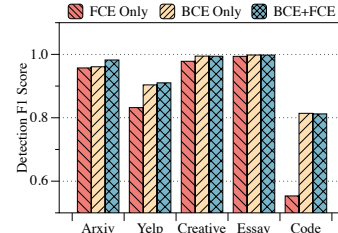

**Figure 5: Comparison with GPTZero on five datasets using five latest commercial generative AI models. The metric used in the figure is the detection F1 score.**

**Figure 6: Comparison of time costs for processing a single sample between BISCOPE and nine baselines.**

**Figure 7: Comparison of BIS-COPE's effectiveness and efficiency when using different open-source LLMs.**

**Figure 8: Comparison of the contributions of $\mathcal{FCE}$ and $\mathcal{BCE}$ to detection effectiveness in BISCOPE.**

baselines with a 0.29 average F1 score increase. Additionally, existing baselines experience an overall 0.03 F1 detection score drop compared to their performance on the normal dataset. In contrast, our BISCOPE performs even better on the paraphrased dataset, with an overall 0.02 average F1 score increase. Under the out-of-distribution setting, we train the classifiers for all the methods on the normal dataset, while testing them on the paraphrased dataset, exploring the generality of the detection against unseen paraphrased data. Our BISCOPE outperforms existing baselines with an average 0.29 F1 detection score increase.

### 4.4 Comparison with The Latest Commercial Detection Method

We also compare BISCOPE with the latest version (`2024-01-09`) of the most renowned commercial AI-generated text detection API, GPTZero [43], across all five datasets, as shown in Figure 5. BISCOPE outperforms GPTZero in 72% of the cases. Specifically, BISCOPE achieves a 0.02, 0.01, and 0.01 average F1 detection score increase on the Arxiv, Essay, and Creative datasets, respectively. On the Yelp dataset, BISCOPE's F1 detection score is 0.04 lower than GPTZero's. However, on the code dataset, BISCOPE performs significantly better than GPTZero, achieving a 0.19 average F1 score improvement, demonstrating BISCOPE's superior generality from natural language to code. Note that GPTZero's detection model is pre-trained on millions of data points, while our BISCOPE's classifier is trained on at most 4,000 test samples for each case.

### 4.5 Efficiency Analysis

To address real-world challenges, efficiency is crucial for detection methods. We compared BISCOPE with nine baselines across five datasets, detailing average processing times for a single sample in Figure 6. RADAR is the fastest at 0.01s per sample, while its adversarial training overhead may be considerable. Processing times for the other baselines are as follows: zero-shot query at 0.32s, LogRank at 0.05s, LRR at 0.10s, DetectGPT at 15.95s, Raidar at 27.42s, OpenAI Detector at 0.03s, Binoculars at 0.19s, and GhostBuster at 0.37s. Zero-shot query, LogRank, LRR, and OpenAI Detector are much quicker than DetectGPT, Raidar, Binoculars, and GhostBuster but offer lower detection performance and robustness. Our BISCOPE processes a sample in 0.14s without summary guidance, matching the real-time levels of zero-shot query, LogRank, and LRR, while improving detection F1 score by over 0.30. With summary guidance, BISCOPE's processing time increases to 1.35s per

sample, still 12 to 20 times faster than DetectGPT and Raidar, and achieves the highest detection score.

## 4.6 Ablation Study

We further evaluate the importance of each component in BıSCOPE with two categories of ablation experiments. Since BıSCOPE utilizes six open-source LLMs in parallel and ensemble their features, we first investigate the contribution of each LLM's feature in Figure 7 individually. Then, we further explore the contribution of BıSCOPE's $\mathcal{FCE}$ and $\mathcal{BCE}$ losses respectively, compared with their aggregated performance, shown as Figure 8. More detailed results are shown in Appendix E. We also present more detailed ablation study on the impact of different segmentation strategies in multi-point splitting (Appendix E.3), and the impact of the completion prompt (Appendix E.4), in Appendix E.

**BıSCOPE's Performance with Different Base Models.** We individually test all the detection base models in BıSCOPE both with and without the summary procedure, including Gemma-2B, Gemma-7B, Llama-2-7B, Mistral-7B, Llama-3-8B, and Llama-2-13B. All the detection models help BıSCOPE achieve over a $0.84$ overall F1 detection score across all five datasets, demonstrating high consistency across different detection base models. As the size of the detection base model increases, BıSCOPE performs progressively better, with F1 scores improving from $0.84$ to $0.95$.

**Importance of $\mathcal{FCE}$ and $\mathcal{BCE}$ in BıSCOPE's Detection.** To demonstrate the rationality of the proposed bi-directional cross-entropy losses, we also evaluate BıSCOPE by only using either $\mathcal{FCE}$ or $\mathcal{BCE}$ losses, and using both of them with the Llama-2-7B model, as shown in Figure 8. When using both the $\mathcal{FCE}$ and $\mathcal{BCE}$ losses, BıSCOPE achieves the best average F1 detection score across the five datasets, which is $0.94$, highlighting the necessity of combining both $\mathcal{FCE}$ and $\mathcal{BCE}$ loss features. When only using $\mathcal{FCE}$ or $\mathcal{BCE}$, BıSCOPE reaches $0.86$ and $0.93$ average F1 detection scores, respectively, indicating that $\mathcal{BCE}$ is more discriminative than $\mathcal{FCE}$.

## 5 Limitations and Future Work

Our BıSCOPE can achieve over a $0.95$ F1 detection score across five datasets generated by the five latest commercial LLMs, both with and without intentional paraphrasing, illustrating the importance of the preceding token information in the output logits. However, in the OOD cross-dataset setting, there is a noticeable $> 0.10$ detection F1 score drop, highlighting the challenges in this setting. Thus, there is still room for future research to achieve more effective and robust detection in few-shot and cross-dataset settings, where the training set contains fewer samples and the test set comes from different text domains. Additionally, exploring ways to further exploit preceding token information and combine it with next token information should also be a future direction in the field of AI-generated text detection.

## 6 Conclusion

Existing methods to differentiate AI-generated texts from human-generated texts often analyze the difficulty for a surrogate LLM to generate the next token based on previous tokens from the text. We propose a more discriminative approach via a novel bi-directional cross-entropy calculation method, leveraging both the preceding token information and the next token information in the output logits. We integrate this method into a four-step detection pipeline, BıSCOPE, which consists of Completion Prompt Generation, Loss Computation in Text Completion, Statistical Feature Extraction, and Feature Classification. We evaluate BıSCOPE on five datasets, including both natural language and code, against six existing detection methods. BıSCOPE surpasses all these methods, improving the average F1 detection score by $0.30$ and also outperforming the well-known commercial API – GPTZero in 72% cases, while maintaining a real-time processing speed of less than 200ms per sample.

## Acknowledgments and Disclosure of Funding

We are grateful to the Center for AI Safety for providing computational resources. This work was funded in part by the National Science Foundation (NSF) Awards SHF-1901242, SHF-1910300, Proto-OKN 2333736, IIS-2416835, DARPA VSPELLS - HR001120S0058, IARPA TrojAI W911NF-19-S0012, ONR N000141712045, N000141410468 and N000141712947. Any opinions, findings and conclusions or recommendations expressed in this material are those of the authors and do not necessarily reflect the views of the sponsors.

## References

[1] Josh Achiam, Steven Adler, Sandhini Agarwal, Lama Ahmad, Ilge Akkaya, Florencia Leoni Aleman, Diogo Almeida, Janko Altenschmidt, Sam Altman, Shyamal Anadkat, et al. Gpt-4 technical report. *arXiv preprint arXiv:2303.08774*, 2023.

[2] Anthropic. Claude 3 api. *https://www.anthropic.com/news/claude-3-family*, 2023.

[3] Anton Bakhtin, Sam Gross, Myle Ott, Yuntian Deng, Marc'Aurelio Ranzato, and Arthur Szlam. Real or fake? learning to discriminate machine from human generated text. *arXiv preprint arXiv:1906.03351*, 2019.

[4] Guangsheng Bao, Yanbin Zhao, Zhiyang Teng, Linyi Yang, and Yue Zhang. Fast-detectgpt: Efficient zero-shot detection of machine-generated text via conditional probability curvature. In *International Conference on Learning Representations (ICLR)*, 2024.

[5] Amrita Bhattacharjee and Huan Liu. Fighting fire with fire: can chatgpt detect ai-generated text? *ACM SIGKDD Explorations Newsletter*, 25(2):14–21, 2024.

[6] Sid Black, Leo Gao, Phil Wang, Connor Leahy, and Stella Biderman. GPT-Neo: Large Scale Autoregressive Language Modeling with Mesh-Tensorflow, 2021.

[7] Jack T Brassil, Steven Low, Nicholas F Maxemchuk, and Lawrence O'Gorman. Electronic marking and identification techniques to discourage document copying. *IEEE Journal on Selected Areas in Communications (JSAC)*, 13(8):1495–1504, 1995.

[8] Mark Chen, Jerry Tworek, Heewoo Jun, Qiming Yuan, Henrique Ponde de Oliveira Pinto, Jared Kaplan, Harri Edwards, Yuri Burda, Nicholas Joseph, Greg Brockman, et al. Evaluating large language models trained on code. *arXiv preprint arXiv:2107.03374*, 2021.

[9] Xiuyuan Cheng and Alexander Cloninger. Classification logit two-sample testing by neural networks for differentiating near manifold densities. *IEEE Transactions on Information Theory*, 68(10):6631–6662, 2022.

[10] Debby RE Cotton, Peter A Cotton, and J Reuben Shipway. Chatting and cheating: Ensuring academic integrity in the era of chatgpt. *Innovations in Education and Teaching International*, 61(2):228–239, 2024.

[11] Sebastian Gehrmann, Hendrik Strobelt, and Alexander Rush. GLTR: Statistical detection and visualization of generated text. In *Annual Meeting of the Association for Computational Linguistics: System Demonstrations (ACL)*, 2019.

[12] Biyang Guo, Xin Zhang, Ziyuan Wang, Minqi Jiang, Jinran Nie, Yuxuan Ding, Jianwei Yue, and Yupeng Wu. How close is chatgpt to human experts? comparison corpus, evaluation, and detection. *arXiv preprint arXiv:2301.07597*, 2023.

[13] Abhimanyu Hans, Avi Schwarzschild, Valeriia Cherepanova, Hamid Kazemi, Aniruddha Saha, Micah Goldblum, Jonas Geiping, and Tom Goldstein. Spotting llms with binoculars: Zero-shot detection of machine-generated text. In *International Conference on Machine Learning (ICML)*, 2024.

[14] Xinlei He, Xinyue Shen, Zeyuan Chen, Michael Backes, and Yang Zhang. Mgtbench: Benchmarking machine-generated text detection. *arXiv preprint arXiv:2303.14822*, 2023.

[15] Abe Hou, Jingyu Zhang, Tianxing He, Yichen Wang, Yung-Sung Chuang, Hongwei Wang, Lingfeng Shen, Benjamin Van Durme, Daniel Khashabi, and Yulia Tsvetkov. Semstamp: A semantic watermark with paraphrastic robustness for text generation. In *Conference of the North American Chapter of the Association for Computational Linguistics: Human Language Technologies (NAACL)*, 2024.

[16] Xiaomeng Hu, Pin-Yu Chen, and Tsung-Yi Ho. Radar: Robust ai-text detection via adversarial learning. *Advances in Neural Information Processing Systems (NeurIPS)*, 2023.

[17] Daphne Ippolito, Daniel Duckworth, Chris Callison-Burch, and Douglas Eck. Automatic detection of generated text is easiest when humans are fooled. In *Annual Meeting of the Association for Computational Linguistics: System Demonstrations (ACL)*, 2020.

[18] Ganesh Jawahar, Muhammad Abdul Mageed, and VS Laks Lakshmanan. Automatic detection of machine generated text: A critical survey. In *International Conference on Computational Linguistics (COLING)*, 2020.

[19] Mohan S Kankanhalli and KF Hau. Watermarking of electronic text documents. *Electronic Commerce Research*, 2:169–187, 2002.

[20] John Kirchenbauer, Jonas Geiping, Yuxin Wen, Jonathan Katz, Ian Miers, and Tom Goldstein. A watermark for large language models. In *International Conference on Machine Learning (ICML)*, 2023.

[21] Ryuto Koike, Masahiro Kaneko, and Naoaki Okazaki. Outfox: Llm-generated essay detection through in-context learning with adversarially generated examples. In *AAAI Conference on Artificial Intelligence (AAAI)*, 2024.

[22] Sarah Kreps, R Miles McCain, and Miles Brundage. All the news that's fit to fabricate: Ai-generated text as a tool of media misinformation. *Journal of Experimental Political Science*, 9(1):104–117, 2022.

[23] Kalpesh Krishna, Yixiao Song, Marzena Karpinska, John Wieting, and Mohit Iyyer. Paraphrasing evades detectors of ai-generated text, but retrieval is an effective defense. *Advances in Neural Information Processing Systems (NeurIPS)*, 2023.

[24] Rohith Kuditipudi, John Thickstun, Tatsunori Hashimoto, and Percy Liang. Robust distortion-free watermarks for language models. *arXiv preprint arXiv:2307.15593*, 2023.

[25] Alex M Lamb, Anirudh Goyal ALIAS PARTH GOYAL, Ying Zhang, Saizheng Zhang, Aaron C Courville, and Yoshua Bengio. Professor forcing: A new algorithm for training recurrent networks. *Advances in Neural Information Processing Systems (NeurIPS)*, 2016.

[26] Junyi Li, Xiaoxue Cheng, Wayne Xin Zhao, Jian-Yun Nie, and Ji-Rong Wen. Halueval: A large-scale hallucination evaluation benchmark for large language models. In *Conference on Empirical Methods in Natural Language Processing (EMNLP)*, 2023.

[27] Jiayi Liang, Xi Zhang, Yuming Shang, Sanchuan Guo, and Chaozhuo Li. Clean-label poisoning attack against fake news detection models. In *IEEE International Conference on Big Data (BigData)*, 2023.

[28] Feng Liu, Wenkai Xu, Jie Lu, Guangquan Zhang, Arthur Gretton, and Danica J Sutherland. Learning deep kernels for non-parametric two-sample tests. In *International Conference on Machine Learning (ICML)*, 2020.

[29] Yinhan Liu, Myle Ott, Naman Goyal, Jingfei Du, Mandar Joshi, Danqi Chen, Omer Levy, Mike Lewis, Luke Zettlemoyer, and Veselin Stoyanov. Roberta: A robustly optimized bert pretraining approach. *arXiv preprint arXiv:1907.11692*, 2019.

[30] Chung Kwan Lo. What is the impact of chatgpt on education? a rapid review of the literature. *Education Sciences*, 13(4):410, 2023.

[31] David Lopez-Paz and Maxime Oquab. Revisiting classifier two-sample tests. In *International Conference on Learning Representations (ICLR)*, 2016.

[32] Chengzhi Mao, Carl Vondrick, Hao Wang, and Junfeng Yang. Raidar: generative ai detection via rewriting. *International Conference on Learning Representations (ICLR)*, 2024.

[33] Hope McGovern, Rickard Stureborg, Yoshi Suhara, and Dimitris Alikaniotis. Your large language models are leaving fingerprints. *arXiv preprint arXiv:2405.14057*, 2024.

[34] Niloofar Mireshghallah, Justus Mattern, Sicun Gao, Reza Shokri, and Taylor Berg-Kirkpatrick. Smaller language models are better zero-shot machine-generated text detectors. In *Conference of the European Chapter of the Association for Computational Linguistics (EACL)*, 2024.

[35] Eric Mitchell, Yoonho Lee, Alexander Khazatsky, Christopher D Manning, and Chelsea Finn. Detectgpt: Zero-shot machine-generated text detection using probability curvature. In *International Conference on Machine Learning (ICML)*, 2023.

[36] OpenAI. Gpt-3.5 turbo api. *https://platform.openai.com/docs/models/gpt-3-5-turbo*, 2023.

[37] Mike Perkins. Academic integrity considerations of ai large language models in the post-pandemic era: Chatgpt and beyond. *Journal of University Teaching & Learning Practice*, 20(2):07, 2023.

[38] Colin Raffel, Noam Shazeer, Adam Roberts, Katherine Lee, Sharan Narang, Michael Matena, Yanqi Zhou, Wei Li, and Peter J. Liu. Exploring the limits of transfer learning with a unified text-to-text transformer. *Journal of Machine Learning Research (JMLR)*, 21(140):1–67, 2020.

[39] Vinu Sankar Sadasivan, Aounon Kumar, Sriram Balasubramanian, Wenxiao Wang, and Soheil Feizi. Can ai-generated text be reliably detected? *arXiv preprint arXiv:2303.11156*, 2023.

[40] Irene Solaiman, Miles Brundage, Jack Clark, Amanda Askell, Ariel Herbert-Voss, Jeff Wu, Alec Radford, Gretchen Krueger, Jong Wook Kim, Sarah Kreps, et al. Release strategies and the social impacts of language models. *arXiv preprint arXiv:1908.09203*, 2019.

[41] Jinyan Su, Terry Yue Zhuo, Di Wang, and Preslav Nakov. Detectllm: Leveraging log rank information for zero-shot detection of machine-generated text. *Conference on Empirical Methods in Natural Language Processing (EMNLP)*, 2023.

[42] Gemini Team, Rohan Anil, Sebastian Borgeaud, Yonghui Wu, Jean-Baptiste Alayrac, Jiahui Yu, Radu Soricut, Johan Schalkwyk, Andrew M Dai, Anja Hauth, et al. Gemini: A family of highly capable multimodal models. *arXiv preprint arXiv:2312.11805*, 2023.

[43] Edward Tian and Alexander Cui. Gptzero: Towards detection of ai-generated text using zero-shot and supervised methods", 2023.

[44] Umut Topkara, Mercan Topkara, and Mikhail J Atallah. The hiding virtues of ambiguity: quantifiably resilient watermarking of natural language text through synonym substitutions. In *The 8th Workshop on Multimedia and Security*, 2006.

[45] Eduard Tulchinskii, Kristian Kuznetsov, Laida Kushnareva, Daniil Cherniavskii, Sergey Nikolenko, Evgeny Burnaev, Serguei Barannikov, and Irina Piontkovskaya. Intrinsic dimension estimation for robust detection of ai-generated texts. *Advances in Neural Information Processing Systems (NeurIPS)*, 2023.

[46] Honai Ueoka, Yugo Murawaki, and Sadao Kurohashi. Frustratingly easy edit-based linguistic steganography with a masked language model. In *The North American Chapter of the Association for Computational Linguistics: Human Language Technologies (NAACL)*, 2021.

[47] Ashish Vaswani, Noam Shazeer, Niki Parmar, Jakob Uszkoreit, Llion Jones, Aidan N Gomez, Łukasz Kaiser, and Illia Polosukhin. Attention is all you need. *Advances in Neural Information Processing Systems (NeurIPS)*, 2017.

[48] Vivek Verma, Eve Fleisig, Nicholas Tomlin, and Dan Klein. Ghostbuster: Detecting text ghostwritten by large language models. *arXiv preprint arXiv:2305.15047*, 2023.

[49] Eric Wallace, Tony Zhao, Shi Feng, and Sameer Singh. Concealed data poisoning attacks on nlp models. In *Conference of the North American Chapter of the Association for Computational Linguistics: Human Language Technologies (NAACL)*, 2021.

[50] Yuxia Wang, Jonibek Mansurov, Petar Ivanov, Jinyan Su, Artem Shelmanov, Akim Tsvigun, Chenxi Whitehouse, Osama Mohammed Afzal, Tarek Mahmoud, Alham Fikri Aji, et al. M4: Multi-generator, multi-domain, and multi-lingual black-box machine-generated text detection. *arXiv preprint arXiv:2305.14902*, 2023.

[51] Wilson Wu, John X Morris, and Lionel Levine. Do language models plan ahead for future tokens? *arXiv preprint arXiv:2404.00859*, 2024.

[52] Xi Yang, Kejiang Chen, Weiming Zhang, Chang Liu, Yuang Qi, Jie Zhang, Han Fang, and Nenghai Yu. Watermarking text generated by black-box language models. *arXiv preprint arXiv:2305.08883*, 2023.

[53] Xianjun Yang, Wei Cheng, Linda Petzold, William Yang Wang, and Haifeng Chen. Dna-gpt: Divergent n-gram analysis for training-free detection of gpt-generated text. *International Conference on Learning Representations (ICLR)*, 2024.

[54] Shuhai Zhang, Yiliao Song, Jiahao Yang, Yuanqing Li, Bo Han, and Mingkui Tan. Detecting machine-generated texts by multi-population aware optimization for maximum mean discrepancy. In *International Conference on Learning Representations (ICLR)*, 2024.

[55] Yue Zhang, Yafu Li, Leyang Cui, Deng Cai, Lemao Liu, Tingchen Fu, Xinting Huang, Enbo Zhao, Yu Zhang, Yulong Chen, et al. Siren's song in the ai ocean: a survey on hallucination in large language models. *arXiv preprint arXiv:2309.01219*, 2023.

To further illustrate our BISCOPE with more details, we present the following materials in the Appendix, shown as follows:

## A    Hyper-parameter Settings

We use the default best-performance settings for all the baselines. Specifically, we use GPT-Neo-2.7B [6] as the surrogate model in LogRank and LRR, and as the scoring model in DetectGPT, where we use T5-3B [38] as the mask-filling model. For RADAR, we use its officially released detection model, which is a pre-trained RoBERTa-Large [29] model. For Zero-shot Query and Raidar, we use GPT-3.5-Turbo as the query model and the rewriting model. For other baselines, we strictly follow their official implementations with their best configurations. For our BISCOPE, we utilize six open-source LLMs in parallel, including Gemma-2B, Gemma-7B, Llama-2-7B, Mistral-7B, Llama-3-8B, and Llama-2-13B. We also test BISCOPE with (BISCOPE *) and without (BISCOPE) text summary guidance. For better reproducibility, we recommend using either Llama2-7B or Llama2-13B as the surrogate model, while the ensemble of more surrogate models is always welcomed. For the text split method, we recommend splitting the text at every 10% length, as used in our paper.

## B    Additional Motivation Example

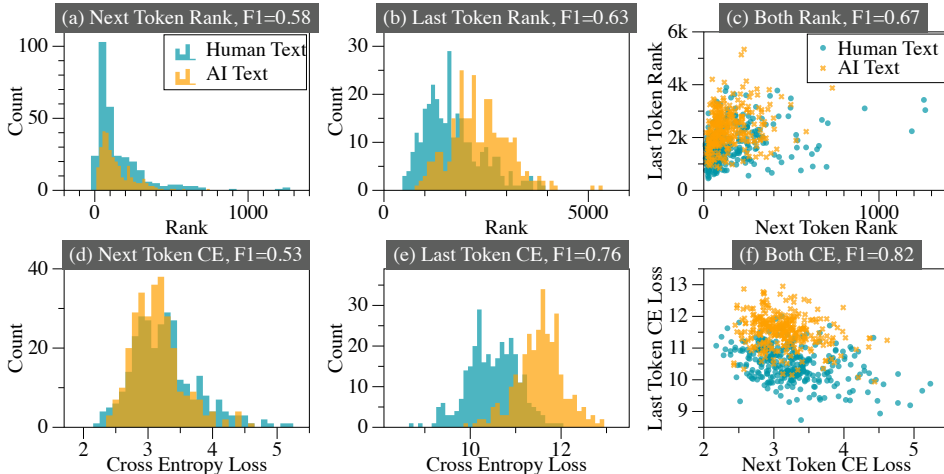

Figure 9: **Comparison between the detection F1 scores when utilizing the rank and cross-entropy loss regarding next token, last token or both. The detection sarrogate model is GPT-Neo 2.7B.**

Figure 9 presents an additional example using a smaller language model: GPT-Neo-2.7B. Specifically, Figure 9(a) and (d) show the detection F1 score when only using the next token's rank and cross-entropy loss for detection. The F1 detection scores are less than 0.60 in both cases. However, when we use the preceding token's rank or cross-entropy loss as the feature, shown in Figure 9(b) and (e), the detection F1 scores increase to 0.63 and 0.76 respectively, indicating the higher reference value of the preceding token information. Figure 9(c) and (f) also show that using both the preceding token information and the next token information in the output logits achieves the best detection performance, with overall detection F1 scores of 0.67 and 0.82.

Moreover, when considering the preceding token information, AI text has a smaller rank score or higher cross-entropy loss, representing worse memorization of the AI text in the output logits.

Conversely, when considering the next token information, the result is the opposite, showing better prediction for AI text. This trend aligns with the results in Figure 1, demonstrating the generality of our observation.

## C  Additional Details about Datasets

**Table 2: Statistical details of the five datasets used in our paper.**

| | | Normal Dataset | | | | | Paraphrased Dataset | | | | |
|---|---|---|---|---|---|---|---|---|---|---|---|
| | Data Type | Dataset Size | Average Len. | Min Len. | Max Len. | Median Len. | Dataset Size | Average Len. | Min Len. | Max Len. | Median Len. |
| Arxiv | Human | 350 | 786.7 | 132 | 1736 | 715.0 | - | - | - | - | - |
| | Machine | 1750 | 787.1 | 101 | 1701 | 810.5 | 1400 | 875.8 | 174 | 1874 | 931.5 |
| | All | 2100 | 787.0 | 101 | 1736 | 799.5 | 1400 | 875.8 | 174 | 1874 | 931.5 |
| Code | Human | 164 | 631.5 | 132 | 1993 | 572.0 | - | - | - | - | - |
| | Machine | 819 | 413.3 | 41 | 1908 | 352.0 | 656 | 493.6 | 13 | 2333 | 382.5 |
| | All | 983 | 449.7 | 41 | 1993 | 387.0 | 656 | 493.6 | 13 | 2333 | 382.5 |
| Yelp | Human | 2000 | 554.9 | 37 | 4959 | 407.0 | - | - | - | - | - |
| | Machine | 9740 | 461.1 | 10 | 2548 | 414.0 | 8000 | 586.5 | 54 | 2593 | 537.0 |
| | All | 11740 | 477.1 | 10 | 4959 | 413.0 | 8000 | 586.5 | 54 | 2593 | 537.0 |
| Essay | Human | 1000 | 4249.9 | 1276 | 41470 | 3301.5 | - | - | - | - | - |
| | Machine | 4897 | 3827.7 | 515 | 21094 | 3486.0 | 3999 | 3666.8 | 129 | 19878 | 3284.0 |
| | All | 5897 | 3899.3 | 515 | 41470 | 3449.0 | 3999 | 3666.8 | 129 | 19878 | 3284.0 |
| Creative | Human | 1000 | 2899.0 | 499 | 9933 | 2462.5 | - | - | - | - | - |
| | Machine | 4840 | 2851.9 | 176 | 13716 | 2620.0 | 4000 | 2924.4 | 85 | 16812 | 2674.5 |
| | All | 5840 | 2860.0 | 176 | 13716 | 2588.5 | 4000 | 2924.4 | 85 | 16812 | 2674.5 |

For all datasets, we reuse their human-generated data and craft AI-generated text using five of the latest commercial LLMs: GPT-3.5-Turbo [36], GPT-4-Turbo [1], Claude-3-Sonnet [2], Claude-3-Opus [2], and Gemini-1.0-Pro [42]. Thus, for each dataset, we have one piece of human-generated data and five pieces of AI-generated data. Additionally, we also create a corresponding paraphrased dataset using similar paraphrasing prompts from previous studies [32] on our newly crafted dataset to evaluate the robustness of the detection methods.

Table 2 presents more detailed information about the datasets used in our paper, for both the normal dataset and the paraphrased dataset. For the short natural language datasets (i.e., Arxiv and Yelp), the average text lengths are 187 and 477, respectively, with minimal lengths of 101 and 10. In contrast, the average text lengths of the long natural language datasets (i.e., Creative and Essay) are 2860 and 3899, which are 6-15 times larger than the short natural language dataset's sample length. The code dataset contains approximately 164 pieces of Python code for human-generated text and AI-generated text from each LLM, with an average length of 983, a minimum length of 41, and a maximum length of 1993. For all the datasets, the human-generated data and AI-generated data share similar lengths, preventing simple length-based detection.

## D  Additional Comparison Results under OOD Setting

Table 3 illustrates more detailed OOD results on both the normal and paraphrased datasets across five datasets. Under the cross-model setting on the normal dataset, BISCOPE outperforms existing baselines in 21 out of 25 cases. Specifically, BISCOPE reaches over a 0.95 detection F1 score against five generative LLMs on the Arxiv, Creative, and Essay datasets, while the detection F1 scores of existing baselines are usually less than 0.90. On the Yelp dataset, BISCOPE achieves over a 0.90 detection F1 score against all five generative models, while the five baselines reach less than a 0.85 detection F1 score in most cases. On the code dataset, our BISCOPE outperforms all baselines except for Raidar, achieving more than a 0.15 detection F1 score increase. Under the cross-dataset setting on the normal dataset, our BISCOPE performs worse than under the cross-model setting, outperforming existing methods in only 9 out of 25 cases. However, in most cases, BISCOPE is the second-best detection method, only trailing RADAR, which uses a detection model pre-trained on texts from multiple domains. As for the OOD setting on the paraphrased dataset, BISCOPE outperforms the five baselines in 17 out of 20 cases, achieving over a 0.90 detection F1 score compared to the < 0.75 average F1 score of the baselines.

Table 3: Detailed performance comparison on both normal and paraphrased dataset under OOD setting.

| | Method | Normal-Cross Model | | | | | Normal-Cross Dataset | | | | | Paraphrased-OOD | | | |
|---|---|---|---|---|---|---|---|---|---|---|---|---|---|---|---|
| | | GPT-3.5 Turbo | GPT-4 Turbo | Claude-3 Sonnet | Claude-3 Opus | Gemini 1.0-pro | GPT-3.5 Turbo | GPT-4 Turbo | Claude-3 Sonnet | Claude-3 Opus | Gemini 1.0-pro | GPT-3.5 Turbo | GPT-4 Turbo | Claude-3 Sonnet | Claude-3 Opus |
| Arxiv | Log Rank | 0.6723 | 0.7033 | 0.7406 | 0.6805 | 0.6597 | 0.6727 | 0.5779 | 0.6476 | 0.6522 | 0.5590 | 0.5814 | 0.2821 | 0.2604 | 0.3382 |
| | LRR | 0.7920 | 0.7187 | 0.7589 | 0.7156 | 0.6742 | 0.6866 | 0.5238 | 0.6302 | 0.6650 | 0.5592 | 0.0881 | 0.2089 | 0.2904 | 0.3537 |
| | DetectGPT | 0.6667 | 0.6637 | 0.6631 | 0.6637 | 0.6637 | 0.6845 | 0.6926 | 0.7343 | 0.7421 | 0.6921 | 0.6402 | 0.5774 | 0.6622 | 0.6609 |
| | RADAR | 0.5721 | 0.7949 | 0.7799 | 0.7854 | 0.7696 | 0.8315 | **0.7638** | 0.8037 | 0.8116 | **0.8071** | 0.9223 | 0.6546 | 0.7128 | 0.6654 |
| | Raidar | 0.5702 | 0.6975 | 0.7604 | 0.6946 | 0.7693 | 0.5010 | 0.6736 | 0.6841 | 0.6888 | 0.7144 | 0.8303 | 0.7195 | 0.6286 | 0.7297 |
| | OpenAI Detector | 0.5209 | 0.6667 | 0.6670 | 0.6657 | 0.6043 | 0.6917 | 0.6653 | 0.6841 | 0.6679 | 0.5758 | 0.6916 | 0.6654 | 0.6608 | 0.6641 |
| | Binoculars | 0.9396 | 0.9272 | 0.9339 | 0.8988 | 0.8818 | **0.8430** | 0.7495 | **0.8489** | **0.8814** | 0.7766 | 0.4284 | 0.4547 | 0.7406 | 0.7105 |
| | GhostBuster | 0.7684 | 0.9370 | 0.9617 | **0.9561** | 0.9702 | 0.5861 | 0.6942 | 0.7281 | 0.6991 | 0.6982 | **0.9543** | **0.9769** | **0.9562** | **0.9814** |
| | BɪScope | 0.9683 | 0.9496 | 0.9665 | 0.9126 | 0.9617 | 0.5706 | 0.6642 | 0.7548 | 0.7968 | 0.7791 | 0.8610 | 0.7461 | 0.8766 | 0.9298 |
| | BɪScope* | **0.9859** | **0.9807** | **0.9881** | 0.9518 | **0.9810** | 0.8250 | 0.6494 | 0.8285 | 0.7875 | 0.7930 | 0.9044 | 0.7678 | 0.8941 | 0.9296 |
| Yelp | Log Rank | 0.6722 | 0.6722 | 0.6309 | 0.6345 | 0.6774 | 0.6695 | 0.6678 | 0.6836 | 0.6841 | 0.6427 | 0.6752 | 0.6417 | 0.5546 | 0.6318 |
| | LRR | 0.6600 | 0.6601 | 0.6475 | 0.6585 | 0.6686 | 0.6667 | 0.6662 | 0.6664 | 0.6664 | 0.6416 | 0.6379 | 0.6644 | 0.6163 | 0.6564 |
| | DetectGPT | 0.6906 | 0.6817 | 0.6738 | 0.6254 | 0.6976 | 0.7124 | 0.6931 | 0.7363 | 0.7534 | 0.6985 | 0.6379 | 0.6345 | 0.7029 | 0.7085 |
| | RADAR | 0.7129 | 0.7180 | 0.7303 | 0.7164 | 0.6964 | 0.8760 | **0.8325** | 0.8129 | 0.7383 | 0.4254 | 0.7506 | 0.6982 | 0.6620 | 0.7023 |
| | Raidar | 0.8337 | 0.8493 | 0.8594 | 0.8749 | 0.8688 | 0.4310 | 0.7533 | 0.7835 | 0.7879 | 0.6695 | 0.7897 | 0.8886 | 0.6993 | 0.7493 |
| | OpenAI Detector | 0.5876 | 0.6587 | 0.6587 | 0.6425 | 0.6267 | 0.6853 | 0.6669 | 0.6663 | 0.6055 | 0.5299 | 0.7243 | 0.6668 | 0.6668 | 0.5674 |
| | Binoculars | 0.7076 | 0.6669 | 0.7028 | 0.6690 | 0.7188 | 0.8809 | 0.6668 | **0.8978** | **0.9350** | 0.8566 | 0.6195 | 0.6632 | 0.6750 | 0.7148 |
| | GhostBuster | 0.7727 | 0.8176 | 0.8074 | 0.8076 | 0.7821 | 0.3737 | 0.7522 | 0.6575 | 0.7167 | 0.4294 | 0.7970 | 0.8539 | 0.8546 | 0.8754 |
| | BɪScope | 0.9010 | 0.9257 | 0.8890 | 0.8982 | 0.9174 | **0.9189** | 0.8189 | 0.8639 | 0.8441 | **0.8580** | 0.9046 | 0.9522 | **0.9787** | **0.9736** |
| | BɪScope* | **0.9134** | **0.9266** | **0.8956** | **0.9049** | **0.9234** | 0.8912 | 0.8056 | 0.8670 | 0.8460 | 0.8175 | 0.9017 | 0.9517 | **0.9787** | 0.9700 |
| Creative | Log Rank | 0.7438 | 0.7836 | 0.8239 | 0.7965 | 0.8485 | 0.5514 | 0.5614 | 0.5938 | 0.6269 | 0.5785 | 0.5132 | 0.3257 | 0.5324 | 0.7138 |
| | LRR | 0.6695 | 0.6864 | 0.7587 | 0.7277 | 0.7786 | 0.5647 | 0.5837 | 0.5835 | 0.5940 | 0.5601 | 0.3500 | 0.3255 | 0.3605 | 0.5024 |
| | DetectGPT | 0.7484 | 0.7466 | 0.7592 | 0.7545 | 0.7780 | 0.4603 | 0.5679 | 0.5063 | 0.5933 | 0.5797 | 0.5669 | 0.5709 | 0.6951 | 0.6506 |
| | RADAR | 0.8639 | 0.9067 | 0.9112 | 0.8919 | 0.8932 | 0.8052 | 0.7352 | 0.7511 | 0.7853 | 0.7727 | **0.9131** | 0.8659 | 0.9129 | 0.8830 |
| | Raidar | 0.7705 | 0.8417 | 0.8328 | 0.8201 | 0.8106 | 0.7048 | 0.7343 | 0.7933 | 0.7779 | 0.7796 | 0.6467 | 0.5797 | 0.5678 | 0.4543 |
| | OpenAI Detector | 0.4144 | 0.6569 | 0.6567 | 0.6567 | 0.6669 | 0.7120 | 0.6667 | 0.6666 | 0.6667 | 0.6420 | 0.1892 | 0.6669 | 0.6669 | 0.6669 |
| | Binoculars | **0.9600** | 0.9783 | 0.9803 | 0.9776 | 0.9595 | 0.7386 | 0.6786 | 0.6999 | 0.8680 | 0.7248 | 0.8999 | 0.7326 | 0.9089 | 0.9537 |
| | GhostBuster | 0.8597 | 0.9825 | 0.9817 | 0.9615 | 0.9650 | **0.8202** | 0.8058 | 0.8296 | 0.8318 | 0.8156 | 0.7059 | 0.9635 | 0.9778 | 0.9576 |
| | BɪScope | 0.9514 | 0.9949 | **0.9960** | **0.9917** | 0.9890 | 0.6212 | 0.8081 | 0.8522 | 0.8596 | **0.8490** | 0.8981 | 0.9950 | 0.9950 | **0.9945** |
| | BɪScope* | 0.9169 | **0.9955** | 0.9950 | 0.9912 | **0.9914** | 0.6507 | **0.8317** | **0.8759** | **0.8732** | 0.8457 | 0.8192 | **0.9965** | **0.9960** | 0.9935 |
| Essay | Log Rank | 0.8185 | 0.9433 | 0.9234 | 0.8989 | 0.9286 | 0.4447 | 0.3644 | 0.4817 | 0.5211 | 0.4472 | 0.4331 | 0.4073 | 0.7244 | 0.8648 |
| | LRR | 0.8415 | 0.8892 | 0.9128 | 0.9010 | 0.9110 | 0.5060 | 0.3917 | 0.4859 | 0.5429 | 0.4354 | 0.4418 | 0.2645 | 0.6478 | 0.8040 |
| | DetectGPT | 0.9087 | 0.9238 | 0.9163 | 0.9096 | 0.9118 | 0.5452 | 0.5392 | 0.6067 | 0.6467 | 0.5469 | 0.6744 | 0.6604 | 0.8352 | 0.8352 |
| | RADAR | 0.9121 | 0.9450 | 0.9398 | 0.9488 | 0.9473 | **0.8088** | **0.7222** | 0.7492 | 0.7828 | 0.7693 | 0.9113 | 0.8028 | 0.9413 | 0.8978 |
| | Raidar | 0.9239 | 0.9536 | 0.9448 | 0.9307 | 0.9552 | 0.7382 | 0.7202 | 0.7058 | 0.6833 | 0.7203 | 0.8889 | 0.7903 | 0.7399 | 0.7809 |
| | OpenAI Detector | 0.2880 | 0.6604 | 0.6606 | 0.6600 | 0.6662 | 0.5673 | 0.6670 | 0.6670 | 0.6669 | 0.6381 | 0.1228 | 0.6647 | 0.6669 | 0.6656 |
| | Binoculars | **0.9954** | 0.9971 | 0.9973 | 0.9968 | **0.9968** | 0.7499 | 0.5924 | **0.7826** | **0.8415** | 0.7253 | **0.9826** | 0.8205 | 0.9733 | 0.9950 |
| | GhostBuster | 0.9500 | 0.9955 | 0.9931 | 0.9853 | 0.9779 | 0.7691 | 0.6945 | 0.7588 | 0.8306 | **0.8167** | 0.9030 | 0.9639 | 0.9362 | 0.9708 |
| | BɪScope | 0.9825 | **0.9981** | **0.9986** | 0.9988 | 0.9949 | 0.4709 | 0.5208 | 0.5579 | 0.6533 | 0.5252 | 0.8179 | **0.9863** | 0.9717 | 0.9979 |
| | BɪScope* | 0.9653 | 0.9987 | **0.9986** | **0.9991** | 0.9952 | 0.4702 | 0.5305 | 0.6152 | 0.6866 | 0.5319 | 0.7531 | 0.9811 | **0.9847** | **0.9980** |
| Code | Log Rank | 0.6629 | 0.6615 | 0.6629 | 0.6315 | 0.6416 | 0.6851 | 0.4556 | 0.6693 | 0.4969 | 0.3459 | 0.5949 | 0.5148 | 0.4518 | 0.6541 |
| | LRR | 0.6614 | 0.6642 | 0.6656 | 0.6642 | 0.6513 | 0.6654 | 0.6529 | 0.6672 | 0.6668 | 0.5714 | 0.6529 | 0.6667 | 0.6499 | 0.6667 |
| | DetectGPT | 0.6588 | 0.6560 | 0.6533 | 0.6434 | 0.6111 | 0.7056 | 0.6632 | 0.7392 | 0.7592 | 0.1008 | 0.6095 | 0.5975 | 0.6390 | 0.6667 |
| | RADAR | 0.6599 | 0.6670 | 0.6663 | 0.6656 | 0.6673 | 0.8476 | **0.7668** | **0.8062** | **0.8232** | **0.8134** | 0.6667 | 0.6653 | 0.6666 | 0.6653 |
| | Raidar | **0.9359** | **0.9498** | **0.9364** | **0.9277** | 0.6893 | 0.0844 | 0.0806 | 0.0494 | 0.1509 | 0.3236 | 0.8685 | 0.6542 | 0.4409 | 0.6448 |
| | OpenAI Detector | 0.6116 | 0.6609 | 0.6689 | 0.6741 | 0.6680 | 0.5297 | 0.0992 | 0.2784 | 0.5289 | 0.6054 | 0.7293 | 0.6122 | 0.3406 | 0.6247 |
| | Binoculars | 0.5510 | 0.6070 | 0.6656 | 0.6567 | 0.6563 | **0.8481** | 0.7608 | 0.6680 | 0.6724 | 0.6412 | 0.7107 | 0.6656 | 0.6667 | 0.6653 |
| | GhostBuster | 0.5725 | 0.6249 | 0.6750 | 0.6809 | 0.5626 | 0.5253 | 0.4645 | 0.5441 | 0.5404 | 0.4714 | 0.8596 | 0.7773 | 0.5906 | 0.4884 |
| | BɪScope | 0.7752 | 0.7661 | 0.8376 | 0.8012 | **0.8068** | 0.6184 | 0.5843 | 0.5612 | 0.5889 | 0.5947 | 0.9683 | 0.9523 | **0.8987** | 0.7801 |
| | BɪScope* | 0.7721 | 0.7621 | 0.8374 | 0.7906 | 0.7869 | 0.6033 | 0.5769 | 0.5750 | 0.5793 | 0.5930 | **0.9719** | **0.9528** | 0.8829 | **0.8021** |

# E  Additional Details of Ablation Study

## E.1  Results with Different Open-source LLMs

Table 4 shows additional details when different detection models are used by BɪScope on both the normal and paraphrased datasets. In most cases, the ensemble results are better than the F1 scores derived using a single detection model. However, at least 90% of the detection performance can be preserved in most single-detection-model settings. Additionally, the detection F1 score consistently increases with the increasing model size of the detection model.

Among all the detection models, the Llama-2 series performs the best, outperforming its updated version, Llama-3. In contrast, the Gemma series performs the worst. A possible reason is that the Gemma and Llama-3 series models are still in their early development stages and are not fully capable of handling the AI-generated text detection task.

**Table 4: Detailed performance comparison with different detection base models in BISCOPE.**

| | | | Normal | | | | | | Paraphrased | | | |
|---|---|---|---|---|---|---|---|---|---|---|---|---|
| | | Model | GPT-3.5 Turbo | GPT-4 Turbo | Claude-3 Sonnet | Claude-3 Opus | Gemini 1.0-pro | Average | GPT-3.5 Turbo | GPT-4 Turbo | Claude-3 Sonnet | Claude-3 Opus | Average |
| Arxiv | w/o summary | Gemma-2B | 0.8997 | 0.8763 | 0.8587 | 0.9290 | 0.9175 | 0.8962 | 0.8606 | 0.8489 | 0.8338 | 0.8953 | 0.8597 |
| | | Gemma-7B | 0.9530 | 0.9335 | 0.8970 | 0.9679 | 0.9336 | 0.9370 | 0.9037 | 0.8678 | 0.8660 | 0.9210 | 0.8896 |
| | | Llama-2-7b | 0.9827 | **0.9957** | 0.9766 | 0.9855 | **0.9708** | 0.9823 | 0.9360 | 0.9535 | 0.9506 | 0.9796 | 0.9549 |
| | | Mixtral-7B | 0.9710 | 0.9843 | 0.9638 | 0.9855 | 0.9663 | 0.9742 | 0.9511 | 0.9561 | 0.9414 | 0.9724 | 0.9553 |
| | | Llama-3-8B | 0.9685 | 0.9786 | 0.9576 | 0.9841 | 0.9665 | 0.9711 | 0.9102 | 0.9002 | 0.8932 | 0.9405 | 0.9110 |
| | | Llama-2-13B | 0.9634 | 0.9871 | 0.9662 | 0.9914 | 0.9678 | 0.9752 | 0.9307 | 0.9400 | 0.9502 | 0.9826 | 0.9509 |
| | | ensemble | **0.9870** | 0.9928 | **0.9796** | **0.9885** | **0.9708** | **0.9837** | **0.9769** | **0.9800** | 0.9625 | **0.9870** | **0.9766** |
| | w/ summary | Gemma-2B | 0.9223 | 0.9255 | 0.9140 | 0.9568 | 0.9282 | 0.9294 | 0.8282 | 0.8384 | 0.8386 | 0.9078 | 0.8533 |
| | | Gemma-7B | 0.9501 | 0.9438 | 0.9208 | 0.9679 | 0.9357 | 0.9437 | 0.8982 | 0.8838 | 0.8777 | 0.9283 | 0.8970 |
| | | Llama-2-7b | **0.9957** | **0.9971** | **0.9913** | 0.9898 | 0.9798 | 0.9907 | 0.9681 | 0.9785 | **0.9710** | **0.9884** | 0.9765 |
| | | Mixtral-7B | 0.9739 | 0.9798 | 0.9679 | 0.9884 | 0.9711 | 0.9762 | 0.9304 | 0.9524 | 0.9361 | 0.9726 | 0.9479 |
| | | Llama-3-8B | 0.9814 | 0.9785 | 0.9708 | 0.9841 | 0.9725 | 0.9775 | 0.9152 | 0.9175 | 0.8787 | 0.9461 | 0.9144 |
| | | Llama-2-13B | 0.9842 | 0.9856 | 0.9766 | 0.9884 | 0.9723 | 0.9814 | 0.9468 | 0.9526 | 0.9485 | 0.9826 | 0.9576 |
| | | ensemble | 0.9928 | 0.9943 | 0.9869 | **0.9913** | **0.9797** | **0.9890** | **0.9870** | **0.9859** | 0.9593 | **0.9884** | **0.9802** |
| Yelp | w/o summary | Gemma-2B | 0.7400 | 0.7393 | 0.7689 | 0.7508 | 0.7557 | 0.7509 | 0.7368 | 0.7578 | 0.8053 | 0.7853 | 0.7713 |
| | | Gemma-7B | 0.8309 | 0.8730 | 0.8958 | 0.8795 | 0.8817 | 0.8722 | 0.8208 | 0.8764 | 0.8944 | 0.8890 | 0.8702 |
| | | Llama-2-7b | 0.8566 | 0.9002 | 0.9446 | 0.9289 | 0.9209 | 0.9102 | 0.8589 | 0.9233 | 0.9717 | 0.9633 | 0.9293 |
| | | Mixtral-7B | 0.8473 | 0.8939 | 0.9365 | 0.9128 | 0.8949 | 0.8971 | 0.8630 | 0.9101 | 0.9632 | 0.9554 | 0.9229 |
| | | Llama-3-8B | 0.8550 | 0.8863 | 0.9301 | 0.9249 | 0.9026 | 0.8998 | 0.8374 | 0.8900 | 0.9271 | 0.9275 | 0.8955 |
| | | Llama-2-13B | 0.8753 | 0.9220 | 0.9586 | 0.9479 | 0.9360 | 0.9280 | 0.8875 | 0.9316 | 0.9770 | 0.9724 | 0.9421 |
| | | ensemble | **0.9023** | **0.9405** | **0.9652** | **0.9532** | **0.9486** | **0.9420** | **0.9064** | **0.9473** | **0.9814** | **0.9789** | **0.9535** |
| | w/ summary | Gemma-2B | 0.7475 | 0.7544 | 0.7912 | 0.8158 | 0.7660 | 0.7750 | 0.7486 | 0.7544 | 0.8396 | 0.8163 | 0.7897 |
| | | Gemma-7B | 0.8218 | 0.8706 | 0.8800 | 0.8800 | 0.8806 | 0.8666 | 0.8145 | 0.8688 | 0.8854 | 0.8870 | 0.8639 |
| | | Llama-2-7b | 0.8655 | 0.9121 | 0.9490 | 0.9341 | 0.9251 | 0.9172 | 0.8689 | 0.9270 | 0.9750 | 0.9656 | 0.9341 |
| | | Mixtral-7B | 0.8724 | 0.9006 | 0.9413 | 0.9281 | 0.9050 | 0.9095 | 0.8782 | 0.9092 | 0.9624 | 0.9532 | 0.9258 |
| | | Llama-3-8B | 0.8727 | 0.8998 | 0.9393 | 0.9305 | 0.9088 | 0.9102 | 0.8511 | 0.8894 | 0.9416 | 0.9314 | 0.9034 |
| | | Llama-2-13B | 0.8771 | 0.9250 | 0.9533 | 0.9380 | 0.9280 | 0.9243 | 0.8878 | 0.9376 | 0.9800 | 0.9709 | 0.9441 |
| | | ensemble | **0.9010** | **0.9452** | **0.9658** | **0.9570** | **0.9545** | **0.9447** | **0.9102** | **0.9530** | **0.9830** | **0.9757** | **0.9555** |
| Creative | w/o summary | Gemma-2B | 0.9819 | 0.9527 | 0.9424 | 0.9620 | 0.9695 | 0.9617 | 0.9664 | 0.9410 | 0.9284 | 0.9586 | 0.9486 |
| | | Gemma-7B | 0.9880 | 0.9555 | 0.9549 | 0.9739 | 0.9751 | 0.9695 | 0.9786 | 0.9611 | 0.9500 | 0.9699 | 0.9649 |
| | | Llama-2-7b | **0.9985** | 0.9930 | 0.9940 | 0.9915 | 0.9934 | 0.9941 | 0.9925 | 0.9925 | 0.9935 | 0.9940 | 0.9931 |
| | | Mixtral-7B | 0.9970 | 0.9920 | 0.9950 | 0.9920 | 0.9964 | 0.9945 | 0.9915 | 0.9920 | 0.9900 | 0.9900 | 0.9909 |
| | | Llama-3-8B | **0.9985** | 0.9754 | 0.9790 | 0.9910 | 0.9911 | 0.9870 | 0.9720 | 0.9510 | 0.9531 | 0.9740 | 0.9625 |
| | | Llama-2-13B | 0.9980 | 0.9940 | 0.9930 | 0.9915 | 0.9958 | 0.9945 | 0.9935 | **0.9950** | 0.9945 | 0.9945 | 0.9944 |
| | | ensemble | **0.9985** | **0.9950** | **0.9960** | **0.9930** | **0.9964** | **0.9958** | **0.9955** | 0.9945 | **0.9955** | **0.9940** | **0.9949** |
| | w/ summary | Gemma-2B | 0.9875 | 0.9519 | 0.9503 | 0.9672 | 0.9694 | 0.9653 | 0.9675 | 0.9459 | 0.9426 | 0.9654 | 0.9554 |
| | | Gemma-7B | 0.9885 | 0.9545 | 0.9541 | 0.9759 | 0.9786 | 0.9703 | 0.9740 | 0.9680 | 0.9540 | 0.9775 | 0.9684 |
| | | Llama-2-7b | **0.9980** | 0.9925 | 0.9950 | 0.9940 | 0.9935 | 0.9946 | 0.9885 | 0.9925 | 0.9940 | 0.9925 | 0.9919 |
| | | Mixtral-7B | 0.9960 | 0.9930 | 0.9950 | 0.9925 | 0.9922 | 0.9937 | 0.9895 | 0.9899 | 0.9935 | 0.9920 | 0.9912 |
| | | Llama-3-8B | **0.9980** | 0.9790 | 0.9835 | 0.9935 | 0.9917 | 0.9891 | 0.9760 | 0.9604 | 0.9723 | 0.9795 | 0.9721 |
| | | Llama-2-13B | 0.9975 | 0.9950 | 0.9950 | 0.9930 | 0.9958 | 0.9953 | 0.9930 | 0.9940 | 0.9945 | 0.9940 | 0.9939 |
| | | ensemble | 0.9975 | **0.9955** | **0.9955** | **0.9945** | **0.9970** | **0.9960** | **0.9955** | **0.9955** | **0.9950** | **0.9945** | **0.9951** |
| Essay | w/o summary | Gemma-2B | 0.9980 | 0.9830 | 0.9935 | 0.9919 | 0.9950 | 0.9923 | 0.9805 | 0.9543 | 0.9666 | 0.9864 | 0.9720 |
| | | Gemma-7B | 0.9965 | 0.9699 | 0.9855 | 0.9900 | 0.9905 | 0.9865 | 0.9779 | 0.9583 | 0.9721 | 0.9784 | 0.9717 |
| | | Llama-2-7b | 0.9995 | 0.9970 | 0.9965 | 0.9965 | 0.9995 | 0.9978 | 0.9940 | 0.9975 | 0.9970 | 0.9970 | 0.9964 |
| | | Mixtral-7B | **1.0000** | 0.9970 | 0.9940 | 0.9955 | 0.9989 | 0.9971 | 0.9920 | 0.9950 | 0.9839 | 0.9955 | 0.9916 |
| | | Llama-3-8B | 0.9995 | 0.9975 | 0.9965 | 0.9965 | **0.9994** | 0.9979 | 0.9905 | 0.9910 | 0.9849 | 0.9955 | 0.9905 |
| | | Llama-2-13B | 0.9995 | 0.9980 | 0.9980 | 0.9960 | **0.9994** | 0.9982 | 0.9955 | 0.9975 | 0.9955 | 0.9970 | 0.9964 |
| | | ensemble | **1.0000** | **0.9990** | **0.9985** | **0.9970** | **0.9994** | **0.9988** | **0.9965** | **0.9990** | **0.9990** | **0.9980** | **0.9981** |
| | w/ summary | Gemma-2B | 0.9970 | 0.9850 | 0.9935 | 0.9945 | 0.9967 | 0.9933 | 0.9810 | 0.9645 | 0.9779 | 0.9875 | 0.9777 |
| | | Gemma-7B | 0.9980 | 0.9743 | 0.9879 | 0.9900 | 0.9899 | 0.9880 | 0.9820 | 0.9673 | 0.9699 | 0.9834 | 0.9757 |
| | | Llama-2-7b | 0.9995 | 0.9975 | 0.9975 | **0.9975** | 0.9994 | 0.9983 | 0.9950 | 0.9970 | 0.9965 | 0.9980 | **0.9966** |
| | | Mixtral-7B | **1.0000** | 0.9960 | 0.9955 | 0.9960 | 0.9972 | 0.9969 | 0.9940 | 0.9945 | 0.9834 | 0.9945 | 0.9916 |
| | | Llama-3-8B | 0.9990 | 0.9975 | 0.9980 | **0.9975** | **0.9989** | 0.9982 | 0.9935 | 0.9925 | 0.9925 | 0.9980 | 0.9941 |
| | | Llama-2-13B | 0.9985 | 0.9980 | 0.9965 | 0.9955 | 0.9983 | 0.9974 | 0.9955 | 0.9985 | 0.9955 | 0.9970 | **0.9966** |
| | | ensemble | **1.0000** | **0.9990** | **0.9985** | **0.9975** | **0.9989** | **0.9988** | **0.9975** | **0.9990** | **0.9985** | **0.9985** | 0.9984 |
| Code | w/o summary | Gemma-2B | 0.7776 | 0.7238 | 0.6018 | 0.5056 | 0.2674 | 0.5752 | 0.9805 | 0.9543 | 0.9666 | 0.9864 | 0.9720 |
| | | Gemma-7B | 0.7759 | 0.7296 | 0.5441 | 0.4515 | 0.2983 | 0.5599 | 0.9779 | 0.9583 | 0.9721 | 0.9784 | 0.9717 |
| | | Llama-2-7b | 0.9479 | 0.9301 | 0.8275 | 0.5677 | 0.7878 | 0.8122 | 0.9940 | 0.9975 | 0.9970 | 0.9970 | 0.9964 |
| | | Mixtral-7B | 0.9501 | 0.9527 | 0.8396 | 0.5834 | 0.6986 | 0.8049 | 0.9920 | 0.9950 | 0.9839 | 0.9955 | 0.9916 |
| | | Llama-3-8B | 0.8388 | 0.7787 | 0.6202 | 0.5560 | 0.4457 | 0.6479 | 0.9905 | 0.9910 | 0.9849 | 0.9955 | 0.9905 |
| | | Llama-2-13B | 0.9630 | 0.9590 | 0.8215 | **0.6151** | **0.7980** | 0.8313 | 0.9955 | 0.9975 | 0.9955 | 0.9970 | 0.9964 |
| | | ensemble | **0.9665** | **0.9655** | **0.8528** | 0.6069 | 0.7809 | **0.8345** | **0.9965** | **0.9990** | **0.9990** | **0.9980** | **0.9981** |
| | w/ summary | Gemma-2B | 0.7975 | 0.7051 | 0.5590 | 0.4716 | 0.2640 | 0.5594 | 0.9810 | 0.9645 | 0.9779 | 0.9875 | 0.9777 |
| | | Gemma-7B | 0.8008 | 0.7295 | 0.5601 | 0.4562 | 0.3130 | 0.5719 | 0.9820 | 0.9673 | 0.9699 | 0.9834 | 0.9757 |
| | | Llama-2-7b | 0.9504 | 0.9346 | 0.8274 | 0.6304 | **0.7768** | 0.8239 | 0.9950 | 0.9970 | 0.9965 | 0.9980 | 0.9966 |
| | | Mixtral-7B | 0.9474 | 0.9280 | 0.8158 | 0.6556 | 0.7146 | 0.8123 | 0.9940 | 0.9945 | 0.9834 | 0.9945 | 0.9916 |
| | | Llama-3-8B | 0.8592 | 0.7871 | 0.6223 | 0.5536 | 0.4954 | 0.6635 | 0.9935 | 0.9925 | 0.9925 | 0.9980 | 0.9941 |
| | | Llama-2-13B | 0.9426 | 0.9469 | 0.8468 | 0.6070 | 0.8084 | 0.8303 | 0.9955 | 0.9985 | 0.9955 | 0.9970 | 0.9966 |
| | | ensemble | **0.9692** | **0.9586** | **0.8526** | **0.6620** | 0.7741 | **0.8433** | **0.9975** | **0.9990** | **0.9985** | **0.9985** | **0.9984** |

**Table 5: Detailed contribution comparison between $\mathcal{FCE}$ and $\mathcal{BCE}$.**

| Dataset | Method | GPT-3.5-Turbo | GPT-4-Turbo | Claude-3-Sonnet | Claude-3-Opus | Gemini-1.0-pro |
|---|---|---|---|---|---|---|
| | | | Generative AI Model | | | |
| Arxiv | $\mathcal{FCE}$ Only | 0.9281 | 0.9698 | 0.9407 | 0.9827 | 0.9647 |
| | $\mathcal{BCE}$ Only | 0.9524 | 0.9685 | 0.9668 | 0.9740 | 0.9429 |
| | $\mathcal{BCE}+\mathcal{FCE}$ | **0.9827** | **0.9957** | **0.9766** | **0.9855** | **0.9708** |
| Yelp | $\mathcal{FCE}$ Only | 0.7934 | 0.7865 | 0.8834 | 0.8626 | 0.8342 |
| | $\mathcal{BCE}$ Only | 0.8435 | **0.9005** | 0.9396 | 0.9198 | 0.9151 |
| | $\mathcal{BCE}+\mathcal{FCE}$ | **0.8566** | 0.9002 | **0.9446** | **0.9289** | **0.9209** |
| Creative | $\mathcal{FCE}$ Only | 0.9965 | 0.9498 | 0.9799 | 0.9855 | 0.9791 |
| | $\mathcal{BCE}$ Only | 0.9955 | **0.9945** | **0.9950** | **0.9935** | **0.9940** |
| | $\mathcal{BCE}+\mathcal{FCE}$ | **0.9985** | 0.9930 | 0.9940 | 0.9915 | 0.9934 |
| Essay | $\mathcal{FCE}$ Only | 0.9980 | 0.9860 | 0.9929 | 0.9930 | 0.9989 |
| | $\mathcal{BCE}$ Only | **0.9995** | 0.9970 | **0.9975** | 0.9965 | 0.9994 |
| | $\mathcal{BCE}+\mathcal{FCE}$ | **0.9995** | **0.9970** | 0.9965 | **0.9965** | **0.9995** |
| Code | $\mathcal{FCE}$ Only | 0.7849 | 0.6439 | 0.4938 | 0.4628 | 0.3817 |
| | $\mathcal{BCE}$ Only | **0.9496** | **0.9466** | 0.8173 | **0.5747** | 0.7819 |
| | $\mathcal{BCE}+\mathcal{FCE}$ | 0.9479 | 0.9301 | **0.8275** | 0.5677 | **0.7878** |

*(Llama-2-7b w/o summary)*

**Table 6: Ablation results of different segmentation methods in multi-point splitting in BISCOPE.**

| | Method | Normal Dataset | | | | | Paraphrased Dataset | | | | Normal Avg. | Paraphrased Avg. |
|---|---|---|---|---|---|---|---|---|---|---|---|---|
| | | GPT-3.5 Turbo | GPT-4 Turbo | Claude-3 Sonnet | Claude-3 Opus | Gemini 1.0-pro | GPT-3.5 Turbo | GPT-4 Turbo | Claude-3 Sonnet | Claude-3 Opus | | |
| Arxiv | Every 50% Text | 0.9754 | 0.9871 | 0.9635 | 0.9855 | 0.9637 | 0.9587 | 0.9756 | 0.9486 | 0.9767 | 0.9750 | 0.9649 |
| | Every 25% Text | 0.9813 | 0.9914 | 0.9662 | 0.9884 | 0.9690 | 0.9675 | 0.9769 | 0.9622 | 0.9797 | 0.9792 | 0.9716 |
| | Every 10% Text (In Paper) | **0.9870** | **0.9928** | **0.9796** | **0.9885** | **0.9708** | **0.9769** | **0.9800** | **0.9625** | **0.9870** | **0.9837** | **0.9766** |
| Yelp | Every 50% Text | 0.8922 | 0.9314 | 0.9584 | 0.9471 | 0.9337 | 0.8921 | 0.9359 | 0.9779 | 0.9704 | 0.9326 | 0.9441 |
| | Every 25% Text | 0.9002 | 0.9381 | 0.9651 | 0.9527 | 0.9466 | 0.9041 | 0.9452 | **0.9817** | 0.9757 | 0.9405 | 0.9517 |
| | Every 10% Text (In Paper) | **0.9023** | **0.9405** | **0.9652** | **0.9532** | **0.9486** | **0.9064** | **0.9473** | 0.9814 | **0.9789** | **0.9420** | **0.9535** |
| Creative | Every 50% Text | **0.9985** | **0.9960** | 0.9940 | **0.9955** | 0.9958 | 0.9950 | **0.9955** | 0.9935 | 0.9930 | **0.9960** | 0.9943 |
| | Every 25% Text | 0.9980 | 0.9955 | **0.9960** | 0.9930 | **0.9970** | **0.9960** | **0.9955** | 0.9950 | 0.9935 | 0.9959 | **0.9950** |
| | Every 10% Text (In Paper) | **0.9985** | 0.9950 | **0.9960** | 0.9930 | 0.9964 | 0.9955 | 0.9945 | **0.9955** | **0.9940** | 0.9958 | 0.9949 |
| Essay | Every 50% Text | **1.0000** | 0.9990 | 0.9965 | 0.9970 | **0.9994** | **0.9975** | **0.9995** | 0.9975 | 0.9975 | 0.9984 | 0.9980 |
| | Every 25% Text | **1.0000** | **0.9990** | **0.9985** | **0.9980** | **0.9994** | 0.9965 | 0.9990 | 0.9985 | **0.9980** | **0.9990** | 0.9980 |
| | Every 10% Text (In Paper) | **1.0000** | **0.9990** | **0.9985** | 0.9970 | **0.9994** | 0.9965 | 0.9990 | **0.9990** | **0.9980** | 0.9988 | **0.9981** |
| Code | Every 50% Text | 0.8564 | 0.8790 | 0.7706 | 0.5933 | 0.6479 | 0.8798 | 0.8752 | 0.9211 | 0.8427 | 0.7495 | 0.8797 |
| | Every 25% Text | 0.9532 | 0.9333 | 0.8115 | **0.6184** | 0.7088 | 0.9363 | 0.9322 | 0.9470 | 0.8856 | 0.8050 | 0.9252 |
| | Every 10% Text (In Paper) | **0.9665** | **0.9655** | **0.8528** | 0.6069 | **0.7809** | **0.9659** | **0.9464** | **0.9691** | **0.9250** | **0.8345** | **0.9516** |

**Table 7: Performance results when not using the completion prompt in BISCOPE.**

| | Method | Normal Dataset | | | | | Paraphrased Dataset | | | | Normal Avg. | Paraphrased Avg. |
|---|---|---|---|---|---|---|---|---|---|---|---|---|
| | | GPT-3.5 Turbo | GPT-4 Turbo | Claude-3 Sonnet | Claude-3 Opus | Gemini 1.0-pro | GPT-3.5 Turbo | GPT-4 Turbo | Claude-3 Sonnet | Claude-3 Opus | | |
| Arxiv | w/o Completion Prompt | 0.9813 | 0.9914 | 0.9767 | **0.9900** | 0.9692 | 0.9714 | **0.9855** | 0.9621 | 0.9766 | 0.9817 | 0.9739 |
| | w/ Completion Prompt | **0.9870** | **0.9928** | **0.9796** | 0.9885 | **0.9708** | **0.9769** | 0.9800 | **0.9625** | **0.9870** | **0.9837** | **0.9766** |
| Yelp | w/o Completion Prompt | 0.9008 | **0.9471** | 0.9686 | **0.9577** | **0.9523** | 0.9058 | **0.9576** | 0.9872 | **0.9817** | **0.9453** | **0.9581** |
| | w/ Completion Prompt | **0.9023** | 0.9405 | 0.9652 | 0.9532 | 0.9486 | **0.9064** | 0.9473 | 0.9814 | 0.9789 | 0.9420 | 0.9535 |
| Creative | w/o Completion Prompt | 0.9980 | **0.9950** | 0.9950 | 0.9925 | 0.9952 | 0.9945 | **0.9955** | 0.9950 | **0.9940** | 0.9952 | 0.9948 |
| | w/ Completion Prompt | **0.9985** | **0.9950** | **0.9960** | **0.9930** | **0.9964** | **0.9955** | 0.9945 | **0.9955** | **0.9940** | **0.9958** | **0.9949** |
| Essay | w/o Completion Prompt | **1.0000** | **0.9995** | **0.9990** | **0.9975** | **0.9994** | **0.9975** | 0.9990 | 0.9990 | 0.9975 | **0.9991** | **0.9982** |
| | w/ Completion Prompt | **1.0000** | 0.9990 | 0.9985 | 0.9970 | **0.9994** | 0.9965 | **0.9990** | 0.9990 | **0.9980** | 0.9988 | 0.9981 |
| Code | w/o Completion Prompt | **0.9786** | **0.9752** | 0.8327 | **0.6703** | 0.7653 | **0.9720** | **0.9532** | **0.9717** | 0.9210 | **0.8444** | **0.9545** |
| | w/ Completion Prompt | 0.9665 | 0.9655 | **0.8528** | 0.6069 | **0.7809** | 0.9659 | 0.9464 | 0.9691 | **0.9250** | 0.8345 | 0.9516 |

Additionally, when using only a single base model, BISCOPE can achieve a processing speed as fast as 0.03s per sample, outperforming all baseline methods except for RADAR.

### E.2 Detailed Contribution Comparison of Forward and Backward Cross-entropy Losses

Table 5 shows a more concrete comparison between the contributions of $\mathcal{FCE}$ and $\mathcal{BCE}$ to BiScope's detection effectiveness. In 21 of 25 cases, $\mathcal{BCE}$ is more discriminative compared with $\mathcal{FCE}$ (especially on code dataset), providing sufficient justifications for BiScope that introduce the preceding token information into the detection. Besides, In 16 of the 25 cases, the combination of $\mathcal{FCE}$ and $\mathcal{BCE}$ outperforms the $\mathcal{FCE}$ only and $\mathcal{BCE}$ only versions, supporting the necessity to use the bi-directional cross-entropy loss calculation method.

### E.3 Impact of Different Segmentation Strategies in Multi-point Splitting in BiScope

Table 6 presents the ablation results with different segmentation strategies in the multi-point splitting of BiScope. We tested three strategies: splitting at every 50% text length, every 25% text length, and every 10% text length (as used in our paper). The results indicate that a more fine-grained splitting interval generally improves BiScope's performance. However, in a small number of cases, a smaller splitting interval may degrade performance. We choose to use 10% as it achieves the highest detection scores in most cases while reaching low degradation in corner cases.

### E.4 Impact of The Completion Prompt in BiScope

Table 7 presents the comparison results when using and not using the completion prompt in BiScope. The results show that in 25 of 45 cases, using the completion prompt performs better. Additionally, the completion prompt is more compatible with the summary procedure. Thus, we chose to use the completion prompt in BiScope.

